# Hyperbolic Embeddings of Supervised Models

**Richard Nock**
Google Research
richardnock@google.com

**Ehsan Amid**
Google DeepMind
eamid@google.com

**Frank Nielsen**
Sony CS Labs, Inc.
frank.nielsen@acm.org

**Alexander Soen**
RIKEN AIP
Australian National University
alexander.soen@anu.edu.au

**Manfred K. Warmuth**
Google Research
manfred@google.com

## Abstract

Models of hyperbolic geometry have been successfully used in ML for two main tasks: embedding *models* in unsupervised learning (*e.g.* hierarchies) and embedding *data*. To our knowledge, there are no approaches that provide embeddings for supervised models; even when hyperbolic geometry provides convenient properties for expressing popular hypothesis classes, such as decision trees (and ensembles). In this paper, we propose a full-fledged solution to the problem in three independent contributions. The first linking the theory of losses for class probability estimation to hyperbolic embeddings in Poincaré disk model. The second resolving an issue for an easily interpretable, unambiguous embedding of (ensembles of) decision trees in this model. The third showing how to smoothly tweak the Poincaré hyperbolic distance to improve its encoding and visualization properties near the border of the disk, a crucial region for our application, while keeping hyperbolicity. This last step has substantial independent interest as it is grounded in a generalization of Leibniz-Newton's fundamental Theorem of calculus.

## 1 Introduction

Models of hyperbolic geometry have been successfully used to embed hierarchies, *i.e.* tree-based structures [14, 20, 38, 36]. Through the property of low-distortion hyperbolic embeddings [30], symbolic (hierarchies) and numeric (quality metrics) properties of unsupervised learning models can be represented in an accurate and interpretable manner [29]. When it comes to supervised learning, the current trend involves embedding *data* in a hyperbolic space, with models trained on the now hyperbolic data [9, 15, 8, 11]. It is important to note that none of these supervised methods embed *models*. Indeed, the focus of the prior literature is to learn supervised models from hyperbolic data, rather than representing supervised models in hyperbolic geometry. This is in stark contrast to the unsupervised usage described, where the (tree-based) structural properties of unsupervised models are directly exploited for good embeddings. This hints at benefits which can be used in the supervised case, especially for popular supervised models which are (structurally) tree-based.

**Our paper** proposes a full-fledged solution for this problem, in three independent contributions.

**The first** focuses on the numerical part (quality metrics) of the embedding and its link with supervised losses: to provide a natural way of embedding classification and the *confidence* of prediction, we show a link between training with the log-loss (posterior estimation) or logistic loss (real-valued classification) and hyperbolic distances computation in the Poincaré disk.

**The second** focuses on the symbolic part (hierarchies) of the embeddings for a popular kind of supervised of models: decision trees (and ensembles). Unlike for unsupervised learning, we show

that getting an unambiguous embedding of a decision tree requires post-processing the model. Our solution extracts its *monotonic* sub-tree via a new class of supervised models we introduce, monotonic decision trees. This is also convenient for explainability purposes [32].

**The third** and most technical one focuses on visualization and the accuracy of the numerical encoding in the Poincaré disk model. The more accurate / confident is a supervised prediction, the closer to the border of the disk it is represented. Thus, the best models will have their best predictions squeezed close to the border. In addition to being suboptimal for visualization, this region is also documented for being numerically error-prone for hyperbolic distance computation [29]. Two general fixes currently exist: encoding values with (exponentially) more bits or utilizing a trick from Riemannian geometry [19]. As neither is satisfactory for our usage, we propose a third, principled route: like other distances, Poincaré distance is integral. We generalize Leibniz-Newton's fundamental Theorem of calculus using a generalization of classical arithmetic [22]. This generalized "tempered" integral provides a parameter to smoothly alter the properties of the "classical" integral. When defining the Poincarè distance, the tempering controls the hyperbolic constant of the embedding whilst also improving the visualization and numeric accuracy of the embedded models. The generalization of integrals to distances has independent interest as many application in ML rely on integral based distances and distortions, *i.e.,* Bregman divergences, $f$-divergences, integral probability metrics, etc.

Experiments are provided on readily available domains, and all proofs, additional results and additional experiments are given in an Appendix.

## 2 Related work

Models of hyperbolic geometry have been mainly useful to embed hierarchies, *i.e.* tree-based structures [14, 20, 38, 36], with a sustained emphasis on coding size and numerical accuracy [19, 29, 37]. In unsupervised learning and clustering, some applications have sought a simple representation of data on the form of a tree or via hyperbolic projections [7, 6, 17, 34]. Approaches dealing with supervised learning assumes the *data* lies in a hyperbolic space: the output visualized is primarily an embedding of the data itself, with additional details linked to classification of secondary importance, either support vector machines [9], neural nets, logistic regression [15], or (ensembles of) decision trees [8, 11]. We insist on the fact that the aforementioned methods do not represent the *models* in the hyperbolic space, even when those models tree-shaped. The question of embedding classifiers is potentially important to improve the state of the art visualization: in the case of decision trees, popular packages stick to a topological layer (the tree graph) to which various additional information about classification are superimposed but without principled links to the "embedding"[*].

## 3 Basic definitions

Training a supervised model starts with a set (sample) of examples, each of which is a pair $(\boldsymbol{x}, y)$, where $\boldsymbol{x} \in \mathcal{X}$ (a *domain*) and $y \in \{-1, 1\}$ (labels or classes). A decision tree (DT) consists of a binary rooted tree $H$, where at each internal nodes, the two outgoing arcs define a Boolean test over observation variables (see Figure 1, center, for an example); $\mathcal{N}(H)$ is the set of nodes of $H$, $\Lambda(H) \subseteq \mathcal{N}(H)$ is the set of leaf nodes of $H$.

The log-loss is best introduced in the context of the theory of *losses for class probability estimation* (CPE) [28]. We follow the notations of [18]: A CPE loss function, $\ell : \mathcal{Y} \times [0, 1] \to \mathbb{R}$, is

$$\ell(y, p) \quad \doteq \quad [\![y = 1]\!] \cdot \ell_1(p) + [\![y = -1]\!] \cdot \ell_{-1}(p), \tag{1}$$

where $[\![.]\!]$ are Iverson brackets [16]. Functions $\ell_1, \ell_{-1}$ are called *partial* losses. A CPE loss is *symmetric* when $\ell_1(p) = \ell_{-1}(1 - p), \forall p \in [0, 1]$. The *log-loss* is a symmetric CPE loss with $\ell_1^{\text{LOG}}(p) \doteq -\log(p)$. The goal of learning using a CPE loss is to optimize the Bayes risk, $\underline{L}(p) \doteq \inf_\pi \mathsf{E}_{\mathsf{Y} \sim p} \ell(\mathsf{Y}, \pi)$. In the case of the log-loss, it is the criterion used to learn DTs in C4.5 [26]:

$$\underline{L}^{\text{LOG}}(p) \quad = \quad -p \cdot \log p - (1 - p) \cdot \log(1 - p). \tag{2}$$

Models like DTs predict an empirical posterior $\hat{\Pr}[\mathsf{Y} = 1|\mathsf{X}]$ at the leaves: for an observation $\boldsymbol{x}$ reaching leaf $\lambda$, the posterior prediction is the local relative proportion of positive examples in leaf $\lambda$, as estimated by the training sample (noted $p_\lambda^+$). There exists a duality between CPE classification

---

[*]See for instance `https://github.com/parrt/dtreeviz`.

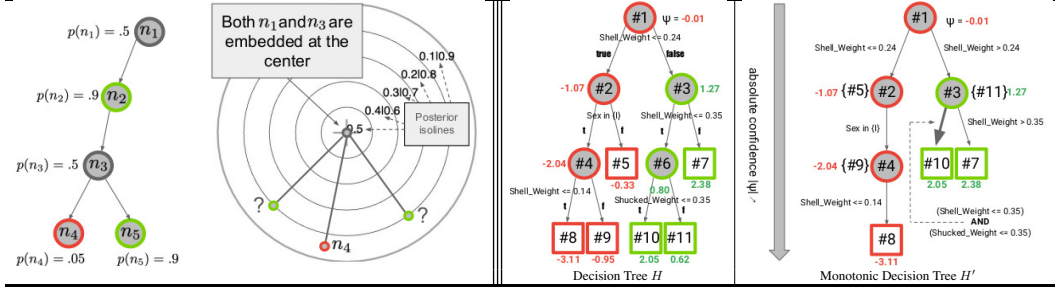

Figure 1: *Left pane*: subtree of a decision tree (DT) (colors red, green denote the majority class in each node, grey = random posterior, tests at arcs not shown, $n_1$ is the root, no leaves shown) and its embedding following the simple recipe in (3): it is impossible to tell $n_2$ from $n_5$ and the tree depicted in $\mathbb{B}$ is not a faithful representation of the DT nor of any of its subtrees. *Right pane*: a small DT learned on UCI `abalone` (left) and its corresponding monotonic decision tree (MDT, right) learned using GETMDT. In each node, the real-valued prediction ($\psi_{\text{LOG}}(p)$) is indicated, also in color. Observe that, indeed, $H$ does not grant path-monotonic classification but $H'$ does (Definition 4.1). In $H'$, some nodes have outdegree 1; also, internal node #6 in the DT, whose prediction is worse than its parent, disappears in $H'$. One arc in $H'$ is represented with double width as its Boolean test aggregates both tests it takes to go from #3 to #10 in $H$. Observations that would be classified by leaf #11 (resp. #5, resp. #9) in $H$ are now classified by internal node #3 (resp. #2, resp. #4) in $H'$, but predicted labels are the same for $H$ and $H'$ and so the accuracy of $H$ and $H'$ are the same.

and the real-valued classification setting familiar to *e.g.* deep learning: optimizing Bayes risk for the posterior is equivalent to minimizing its *convex surrogate* using the *canonical link* of the loss to compute real classification [24, Theorem 1]. The canonical link of the log-loss, $\psi_{\text{LOG}} : [0, 1] \to \mathbb{R}$ is the famed inverse sigmoid, which has a very convenient form for our purpose,

$$\psi_{\text{LOG}}(p) = \log\left(\frac{p}{1-p}\right) \quad \Rightarrow \quad |\psi_{\text{LOG}}(p)| = \log\left(\frac{1+r}{1-r}\right), \quad r \doteq |2p - 1|. \tag{3}$$

Notably, the absolute value $|\psi_{\text{LOG}}(p)|$ is a *confidence* with the sign giving the predicted class. In the case of the log-loss, the convex surrogate is hugely popular in ML: it is the logistic loss.

We now introduce concepts from hyperbolic geometry, particularly those from the Poincaré disk model. Our definitions are simplified, a detailed account can be found in [5, 20]. The distance function $d$ of a metric space is $\tau$-hyperbolic for some $\tau \geqslant 0$ iff for any three geodesic curves $\gamma^1, \gamma^2, \gamma^3$ linking three points, there exists $\boldsymbol{z}$ such that $\max_i d(\boldsymbol{z}, \gamma^i) \leqslant \tau$, where $d(\boldsymbol{z}, \gamma) \doteq \inf_{\boldsymbol{z}' \in \gamma} d(\boldsymbol{z}, \boldsymbol{z}')$. A small hyperbolic constant guarantees thin triangles and embeddings of trees with good properties. The Poincaré disk model, $\mathbb{B}$ (negative curvature, $-1$) is a popular model of hyperbolic geometry with the hyperbolic distance between the origin and some $\boldsymbol{z} \in \mathbb{B}$ with Euclidean norm $r \doteq \|\boldsymbol{z}\|$ being:

$$d_{\mathbb{B}}(\boldsymbol{z}, \boldsymbol{0}) = \log\left(\frac{1+r}{1-r}\right). \tag{4}$$

## 4 Posterior embedding in Poincaré disk and clean embeddings for DTs

**Node embedding** We exploit the similarity between log-loss confidences $|\psi_{\text{LOG}}(p)|$ and hyperbolic distances $d_{\mathbb{B}}(\boldsymbol{z}, \boldsymbol{0})$ when embedding classification models, similarity which is obvious from (3) and (4). In (3), we see that if $r = 0$, the prediction is as bad as a fair coin's. As $r$ edges closer to 1, prediction approaches maximum confidence. The connection with the Poincaré distance (4) is immediate: a natural embedding of a posterior prediction $p$ is then a point $\boldsymbol{z}$ in the Poincaré disk $\mathbb{B}$ at radius $r \doteq \|\boldsymbol{z}\|$ from the origin (3). The origin of Poincaré disk represents the worst possible confidence. As each leaf $\lambda$ of a DT $H$ corresponds to a posterior $p_\lambda^+$, the leaves can be embedded as described. In addition, all nodes $\nu \in \mathcal{N}(H)$ in the tree also have corresponding posteriors, and thus can also be embedded identically to leaf nodes. It is also a good idea to embed them because, as classical DT induction algorithms first proceed by top-down induction, any node $\nu$ in a DT was a leaf at some point during training, and has a corresponding posterior. This apparently clean picture for node embedding however becomes messy as soon as we step to embedding full trees.

**No clean embedding for a full DT** A simple counterexample indeed demonstrates that a clean embedding of a decision tree is substantially trickier, see Figure 1 (left). In this pathological example, some distinct nodes (resp. arcs) of the DT $H$ are embedded in the same node (resp. edge) in $\mathbb{B}$: the depiction has no subgraph relationship to $H$ and some embedded nodes cannot be distinguished between each other without additional information.

**Getting a clean embedding: Monotonic Decision Trees** To introduce our fix, we define three broad objectives for the embedding in $\mathbb{B}$ of DTs

---

Embedding objective

**(A)** Embed a tree-based model from $H$ in $\mathbb{B}$, which defines an injective mapping of the nodes of $H$ and induces a subtree of $H$ with each edge in $\mathbb{B}$ corresponding to a path in $H$;
**(B)** *Locally*, each node $\nu$ of $H$ gets embedded to some $\boldsymbol{z}_\nu$ such that $r_\nu$ is close to $\|\boldsymbol{z}_\nu\|$ (3);
**(C)** *Globally*, the whole embedding remains convenient and readable to compare, in terms of confidence in classification, different subtrees in the same tree, or even between different trees.

---

As we shall see later in this Section, this agenda also carries to embedding ensembles of DTs, thus including their leveraging coefficients, exploiting properties of boosting algorithms. For now, our solution for embedding a single DT that can satisfy (A-C) is simple in principle:

*Embed the monotonic classification part of a DT,*

meaning for each path from the root to a leaf in $H$, we only embed the subsequence of nodes whose absolute confidences are strictly increasing. To do so, we replace the DT by a "close" path-monotonic approximation model using a new class of models we call *Monotonic Decision Trees* (MDT).

**Definition 4.1.** *A Monotonic Decision Tree (MDT) is a rooted tree with a Boolean test labeling each arc and a real-valued prediction at each node. Any sequence of nodes in a path starting from the root is strictly monotonically increasing in absolute confidence. At any internal node, no two Boolean tests at its outgoing arcs can be simultaneously satisfied. The classification of an observation is obtained by the bottom-most node's prediction reachable from the root.*

---

**Algorithm 1** GETMDT($\nu, \mathbf{b}, \nu', \mathbb{I}$)

---

**Input:** Node $\nu \in \mathcal{N}(H)$ ($H$ = DT), Boolean test $\mathbf{b}$, Node $\nu' \in \mathcal{N}(H')$ ($H'$ = MDT being build from $H$), forbidden posteriors $\mathbb{I} \subset [0,1]$;

1 : **if** $\nu \in \Lambda(H)$ **then**
2 :     **if** $p_\nu^+ \in \mathbb{I}$ **then**
3 :         $\nu' \leftarrow$ TAGDTLEAF($\nu$);                            // tags $\nu' \in \mathcal{N}(H')$ with info from leaf $\nu \in \Lambda(H)$
4 :     **else**
5 :         $\nu'' \leftarrow H'$.NEWNODE($\nu$);                        // $\nu''$ will be a new leaf in $H'$
6 :         $H'$.NEWARC($\nu', \mathbf{b}, \nu''$);                         // adds arc $\nu' \to_\mathbf{b} \nu''$ in $H'$
7 :     **endif**
8 : **else**
9 :     **if** $p_\nu^+ \notin \mathbb{I}$ **then**
10 :        $\nu'' \leftarrow H'$.NEWNODE($\nu$);                       // $\nu''$ = new internal node in $H'$
11 :        $H'$.NEWARC($\nu', \mathbf{b}, \nu''$);                        // adds arc $\nu' \to_\mathbf{b} \nu''$ in $H'$
12 :        $\mathbb{I}_{\text{new}} \leftarrow [\min\{p_\nu^+, 1 - p_\nu^+\}, \max\{p_\nu^+, 1 - p_\nu^+\}]$;          // $\mathbb{I} \subset \mathbb{I}_{\text{new}}$
13 :        $\nu'_{\text{new}} \leftarrow \nu''$;
14 :        $\mathbf{b}_{\text{new}} \leftarrow$ **true**;
15 :    **else**
16 :        $\mathbb{I}_{\text{new}} \leftarrow \mathbb{I}; \nu'_{\text{new}} \leftarrow \nu'; \mathbf{b}_{\text{new}} \leftarrow \mathbf{b}$;           // $\nu$ yields no change in $H'$
17 :    **endif**
18 :    GETMDT($\nu$.leftchild, $\mathbf{b}_{\text{new}} \wedge \nu$.TEST(leftchild), $\nu'_{\text{new}}, \mathbb{I}_{\text{new}}$);
19 :    GETMDT($\nu$.rightchild, $\mathbf{b}_{\text{new}} \wedge \nu$.TEST(rightchild), $\nu'_{\text{new}}, \mathbb{I}_{\text{new}}$);
20 : **endif**

---

We introduce an algorithm, GETMDT, which takes as input a DT $H$ and outputs an MDT $H'$, which approximates $H$ via the following key property:

**(M)** For any observation $\boldsymbol{x} \in \mathcal{X}$, the prediction $H'(\boldsymbol{x})$ is equal to the prediction in the path followed by $\boldsymbol{x}$ in $H$ of its deepest node in the strictly monotonic subsequence starting from the root of $H$.

Figure 1 (right) presents an example of MDT $H'$ that would be built for some DT $H$ (left) and satisfying **(M)** (unless observations have missing values, $H'$ is unique). Figure 1 adopts some additional conventions to ease parsing of $H'$, **(D1)** and **(D2)**:

**(D1)** Some internal nodes of $H'$ are also tagged with labels corresponding to the leaves of $H$. If a

node in $H'$ is tagged with a label of one leaf $\lambda$ of $H$, it indicates that examples reaching $\lambda$ in the original $H$ are being classified by $H'$'s tagged node;

**(D2)** Arcs in $H'$ have a width proportional to the number of boolean tests it takes to reach its tail from its head in $H$. A large width thus indicates a long path in $H$ to improve classification confidence.

To produce the MDT $H'$ from DT $H$ with Algorithm 1, after having initialized it to a root = single leaf, we just run

$$\text{GETMDT}(\text{root}(H), \textbf{true}, \text{root}(H'), [\underline{p}^+_{\text{root}}, \overline{p}^+_{\text{root}}])$$

(we let $\underline{p}^+_{\text{root}} \doteq \min\{p^+_{\text{root}}, 1 - p^+_{\text{root}}\}$, $\overline{p}^+_{\text{root}} \doteq \max\{p^+_{\text{root}}, 1 - p^+_{\text{root}}\}$). Upon finishing, the tree rooted at root$(H')$ is the MDT sought.

We complete the description of GETMDT: When a leaf of $H$ does not have sufficient confidence and ends up being mapped to an internal node of the MDT $H'$, TAGDTLEAF is the procedure that tags this internal node with information from the leaf (see **(D1)** above). We consider a tag being just the name of the leaf (Figure 1, leaves $\#5, 9, 11$), but other conventions can be adopted. The other two methods we use grow MDT $H'$ by creating a new node via NEWNODE and adding a new arc between existing nodes via NEWARC. We easily check the following.

**Theorem 1.** $H'$ *built by* GETMDT *satisfies* **(M)** *with respect to DT* $H$.

Property **(M)** is crucial to keep the predictions of the DT and the MDT close to each other: to each node of the MDT indeed corresponds a node in the DT with the same posterior prediction. Because the set of nodes of $H'$ corresponds to a subset of nodes in $H$, predictions can locally change for some observations. This phenomenon is limited by two factors: (i) the set of leaves of the MDT corresponds to a set of leaves in the DT (thus, predictions do not change for the related observations), (ii) more often than not in our experiments, it is only the confidence that changes, not the sign of the prediction, which means that accuracies of the DT and its corresponding MDT are close to each other (Section 6). It is also worth remarking that the MDT $H'$ always has the same depth as the DT $H$. Finally, any pruning of $H$ is a subtree of $H$ and its corresponding MDT is a subtree of the MDT of $H$. The aforementioned troubles to embed the DT in the Poincaré disk considering **(A-C)** do not exist anymore for the MDT because the best embeddings necessarily have all arcs going outwards in the Poincaré disk. The algorithm we use to embed the MDT in Poincaré disk is a simple alteration of Sarkar's embedding [30] which, because of the lack of space, we defer to the Appendix (Section C.VIII).

The quality of the embedding is obtained using an embedding error that takes into account *all* nodes:

$$\rho(H') \quad \doteq \quad 100 \cdot \mathbb{E}_{\nu \sim \mathcal{N}(H')} \left[ \frac{||\psi_\nu| - d_\mathbb{B}(\mathbf{0}, \mathbf{z}_\nu)|}{|\psi_\nu|} \right] \quad (\%), \tag{5}$$

where $\psi_\nu$ refers to the relevant $\psi_{\text{LOG}}(p^+_\nu)$ in (3) (and $\mathbf{z}_\nu \in \mathbb{B}$ is its embedding in $\mathbb{B}$). Two example final representations are in Figure 2. Notice the small errors $\rho$ in both cases.

**Remark 1.** *Using a particular boosting algorithm to boost the logistic loss, we can as well represent the leveraging coefficients of a DT/MDT in a boosted combination, following a convention sketched in Figure 2 and explained at length, along with the boosting algorithm and its properties, in the Appendix (Section C.IX).*

## 5 Smoothly altering integrals: T-calculus and the t-self of Poincaré disk

It is apparent from Figure 2 (right, lowest red pentagon), that when utilizing standard hyperbolic distances $d_\mathbb{B}(\mathbf{z}, \mathbf{0})$ (4), the best MDT nodes are embedded close to the border. In addition to obviously not being great for visualization, these nodes are at risk of having numeric error "push" the nodes to the border $\partial\mathbb{B}$, thereby giving a false depiction of infinite confidence. In this section, we provide alternative distances to prevent this phenomena. First, we quantify the numerical risk, which has been defined by the critical region where high numeric error can occur [19, 29].

**Definition 5.1.** *A point $\mathbf{x} \in \mathbb{B}$ is said $k$-close to boundary $\partial\mathbb{B}$ if $\|\mathbf{x}\| = 1 - 10^{-k}$. It is said encodable iff $\|\mathbf{x}\| < 1$ in machine encoding (it is not "pushed to the boundary").*

Machine encoding constrains the maximum possible $k$: in the double-precision floating-point representation (Float64), $k \approx 16$ [19]. In the case of Poincaré disk, the maximal distance $d_*$ from the

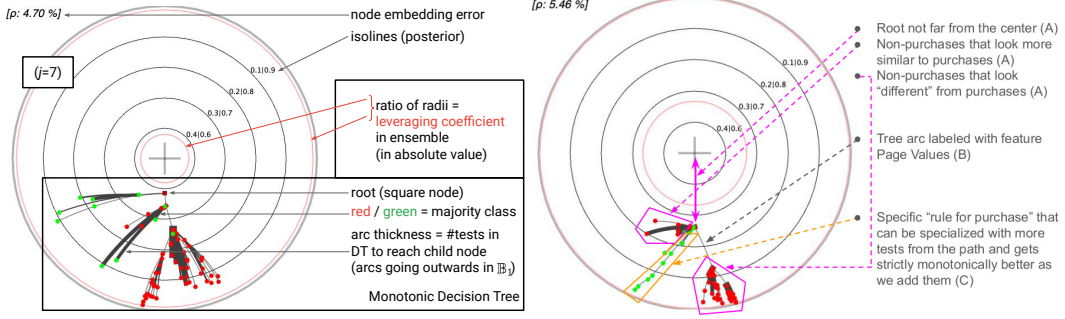

Figure 2: Two example MDTs learned on UCI domain `online_shoppers_intention`, showing embedding error $\rho$ (5). *Left:* key parts of the embedding. *Right:* annotations used in Experiments, Section 6.

origin $\mathbf{0}$ that this authorizes (before numerical error "warps" a point to the border) is a *small* affine order in $k$ [19]:

$$d_* \leqslant \log(2) + \log(10) \cdot k + O(10^{-k}), \tag{6}$$

Hence, in practice, only a ball of radius $d_* \approx 38$ around the origin can be accurately represented. This is in deep contrast with Euclidean representation, where $d_* = \Omega(2^k)$. We now provide a principled solution to changing $d_*$ while keeping hyperbolicity, which relies on a crucial generalization of Leibniz-Newton's fundamental Theorem of calculus.

## 5.1 T-calculus

We let $[n] \doteq \{1, 2, ..., n\}$ for $n \in \mathbb{N}_{>0}$. At the core of our generalization is the replacement of the addition by the tempered (or t-)addition [22], $z \oplus_t z' \doteq z + z' + (1-t)zz'$, in the notion of Riemann sum. The additional term is a scaled saddle point curve $zz'$, positive when the sign of $z$ and $z'$ agree and negative otherwise. Hereafter, $f$ is defined on an interval $[a, b]$. We define a generalization of Riemann integration to t-algebra (see [2] for a preliminary development which does not provide the generalization of Leibniz-Newton's fundamental Theorem of calculus) and for this objective, given an interval $[a, b]$ and a division $\Delta$ of this interval using $n+1$ reals $x_0 \doteq a < x_1 < ... < x_{n-1} < x_n \doteq b$, we define the Riemann t-sum of $f$ over $[a, b]$ using $\Delta$, for a set of tags $\Xi \doteq \{\xi_i \in [x_{i-1}, x_i]\}_{i \in [n]}$,

$$S_\Delta^{(t)}(f) \quad \doteq \quad (\bigoplus_t)_{i \in [n]}(x_i - x_{i-1}) \cdot f(\xi_i) \tag{7}$$

($S_\Delta^{(1)}(f)$ is the classical Riemann summation). Let $s(\Delta) \doteq \max_i x_i - x_{i-1}$ denote the step of division $\Delta$. The conditions for $t$-Riemann integration are the same as for $t = 1$.

**Definition 5.2.** *Fix $t \in \mathbb{R}$. A function $f$ is $t$-(Riemann) integrable over $[a, b]$ iff $\exists L \in \mathbb{R}$ such that $\forall \varepsilon > 0, \exists \delta > 0, \forall$ division $\Delta$ with $s(\Delta) < \delta$ and associated set of tags $\Xi$, $\left| S_\Delta^{(t)}(f) - L \right| < \varepsilon$. When this happens, we note*

$$\overset{(t)}{\int}_a^b f(x)\mathrm{d}_t x = L. \tag{8}$$

The case $t = 1$, Riemann integration, is denoted using classical notations. We now prove our first main results for this Section, namely the link between $t$-Riemann integration and Riemann integration.

**Theorem 2.** *Any function is either $t$-Riemann integrable for all $t \in \mathbb{R}$ simultaneously, or for none. In the former case, we have the relationship: ($\log_t \doteq (z^{1-t} - 1)/(1-t)$ for $t \neq 1$ and $\log_1 \doteq \log$)*

$$\overset{(t)}{\int}_a^b f(u)\mathrm{d}_t u \quad = \quad \mathfrak{q}_t\left(\int_a^b f(u)\mathrm{d}u\right), \forall t \in \mathbb{R}, \quad \text{with } \mathfrak{q}_t(z) \doteq \log_t \exp z. \tag{9}$$

For example, if $t = 0$, we get $1 + \overset{(0)}{\int}_a^b f(u)\mathrm{d}_t u = \exp \int_a^b f(u)\mathrm{d}u$, which is Volterra's integral [33, Theorem 5.5.11]. Theorem 2 is important because it allows to compute any $t$-Riemann integral from

the classical ($t = 1$) case. Classical Riemann integration and derivation are fundamental inverse operations. The classical derivation is sometimes called "Euclidean derivative" [4]. We now elicit the notion of derivative, which is the "inverse" of $t$-Riemann integration. Unsurprisingly, it generalizes Euclidean derivative. The Theorem stands as a generalization of the classical fundamental Theorem of calculus.

**Theorem 3.** *Let $z \ominus_t z' \doteq \frac{z - z'}{1 + (1-t)zz'}$ denote the tempered subtraction. When it exists, define*

$$\mathrm{D}_t f(z) \quad \doteq \quad \lim_{\delta \to 0} \frac{f(z + \delta) \ominus_t f(z)}{\delta}.$$

*Suppose $f$ $t$-Riemann integrable. Then, the function $F$ defined by $F(z) \doteq {}^{(t)}\!\int_a^z f(u)\mathrm{d}_t u$ is such that $\mathrm{D}_t F = f$. We call $F$ a $t$-primitive of $f$ (which zeroes when $z = a$) and $\mathrm{D}_t F$ the $t$-derivative of $F$.*

The function $\mathfrak{q}_t$ (Figure 3) is key to many of our results; it has quite remarkable properties: in particular, it is strictly increasing for any $t \in \mathbb{R}$, strictly concave for any $t > 1$, strictly convex for any $t < 1$ and such that $\mathrm{sign}(\mathfrak{q}_t(z)) = \mathrm{sign}(z), \forall t \in \mathbb{R}$. One can also note that $\mathrm{D}_t \mathfrak{q}_t(z) = 1$. The Appendix proves these results (Sections C.I and C.II).

**Remark 2.** *The Appendix also proves many more results showing how Theorems 2 and 3, and properties of t-calculus, naturally "percolate" through many properties known for classical integration and the fundamental theorem of calculus (among others, additivity, Chasles' relationship, monotonicity, the hyperbolic Pythagorean theorem, mean value theorem, chain rule, etc.).*

## 5.2 The t-self of Poincaré disk

Let us now put T-calculus to good use in our context. For a set $\mathcal{X}$ endowed with function $d : \mathcal{X} \times \mathcal{X} \to [0, +\infty]$ (*e.g.* Poincaré disk and the Poincaré hyperbolic distance), the t-self of $\mathcal{X}$ is the set (implicitly) endowed with $d^{(t)} \doteq \mathfrak{q}_t \circ d$. Before going further, let us provide the elegant $d^{(t)}$ associated to Poincaré disk's $t$-self (with $r \doteq \|\boldsymbol{z}\|$ as in (4)):

$$d_{\mathbb{B}^{(t)}}(\mathbf{0}, \boldsymbol{z}) = \log_t \exp d_{\mathbb{B}}(\mathbf{0}, \boldsymbol{z}) = \log_t \left( \frac{1 + r}{1 - r} \right). \tag{10}$$

The following Lemma (shown in Appendix) demonstrates the usefulness of T-calculus to address the encoding problem in $\mathbb{B}$.

**Lemma 1.** *In Poincaré disk model, pick any increasing function $g(k) \geqslant 0$. For the choice $t = 1 - f(k)$ where $f(k) \in \mathbb{R}$ is any function satisfying*

$$\frac{\log\left(1 + f(k)g(k)\right)}{f(k)} \quad \leqslant \quad \log(10) \cdot k, \tag{11}$$

*we get in lieu of (6) the maximal $d_*^{(t)}$ of $d^{(t)}$ satisfying $d_*^{(t)} \geqslant g(k)$, and the new hyperbolic constant $\tau_t$ of the $t$-self satisfies*

$$\tau_t \quad = \quad \frac{\exp\left(f(k)\tau\right) - 1}{f(k)}. \tag{12}$$

Since both $\log(1 + x) = x + o(x)$ and $\exp(x) - 1 = x + o(x)$ in a neighborhood of 0, we check that (11) and (12) get back to properties of the Poincaré model as $f(k) \to 0$ [19]. We then have two cases: we can easily approach back the Euclidean $d_*$ by picking $f(k) > 0$: choosing $f(k) = 1$ gets there and we still keep a finite hyperbolic constant, albeit exponential in the former one. If however we want to improve further the hyperbolic constant, we can pick some admissible $f(k) < 0$, but then (11) will constrain $g(k)$ to a very small value. Note that depending on the sign of $f(k)$, the triangle inequality can still hold or be replaced by a weaker version since we can show $d_{\mathbb{B}^{(t)}}(\boldsymbol{x}, \boldsymbol{z}) \leqslant d_{\mathbb{B}^{(t)}}(\boldsymbol{x}, \boldsymbol{y}) + d_{\mathbb{B}^{(t)}}(\boldsymbol{y}, \boldsymbol{z}) + \max\{0, f(k)\} \cdot d_{\mathbb{B}^{(t)}}(\boldsymbol{x}, \boldsymbol{y}) \cdot d_{\mathbb{B}^{(t)}}(\boldsymbol{y}, \boldsymbol{z})$ in $\mathbb{B}^{(t)}$.

Let us finally investigate how we can perform a general embedding in the $t$-self $\mathbb{B}^{(t)}$ of the Poincaré disk to be (i) more encoding-savvy and (ii) have points originally close to $\partial\mathbb{B}$ substantially further away from the $t$-self $\mathbb{B}^{(t)}$ border. We also add a third constraint (iii) that the distortions must be *fair w.r.t.* the original the Poincaré disk model embedding; *i.e.* if some $\boldsymbol{z} \in \mathbb{B}$ (*e.g.* the coordinate of a DT node) gets mapped to $\boldsymbol{z}^{(t)} \in \mathbb{B}^{(t)}$ (the t-self), then we request

$$d_{\mathbb{B}^{(t)}}(\boldsymbol{z}^{(t)}, \mathbf{0}) = d_{\mathbb{B}}(\boldsymbol{z}, \mathbf{0}). \tag{13}$$

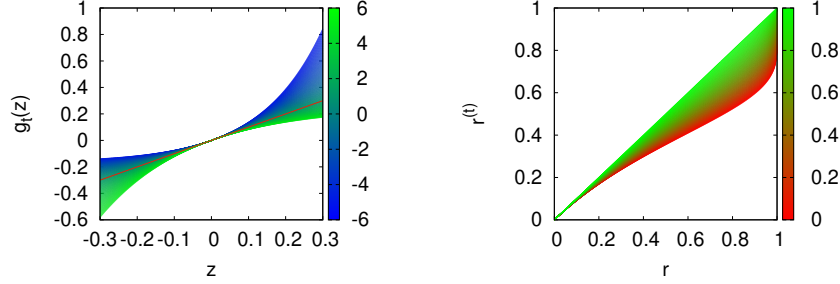

Figure 3: *Left*: plot of $\mathfrak{g}_t(z)$ (9) for different values of $t$ (color map on the right bar), showing where it is convex / concave. The $t = 1$ case ($\mathfrak{g}_1(z) = z$) is emphasized in red. *Right*: suppose $r \doteq \|z\|$ is the Euclidean norm of a point $z$ in Poincaré disk $\mathbb{B}$. Fix $t \in [0, 1]$ (color map on the right bar). The plot gives the (Euclidean) norm $r^{(t)}$ of a point $z^{(t)}$ in the t-self such that $d_{\mathbb{B}^{(t)}}(z^{(t)}, \mathbf{0}) = d_{\mathbb{B}}(z, \mathbf{0})$ (13). Remark that even for $r$ very close to 1 we can have $r^{(t)}$ substantially smaller (*e.g.* $r^{(t)} < 0.8$).

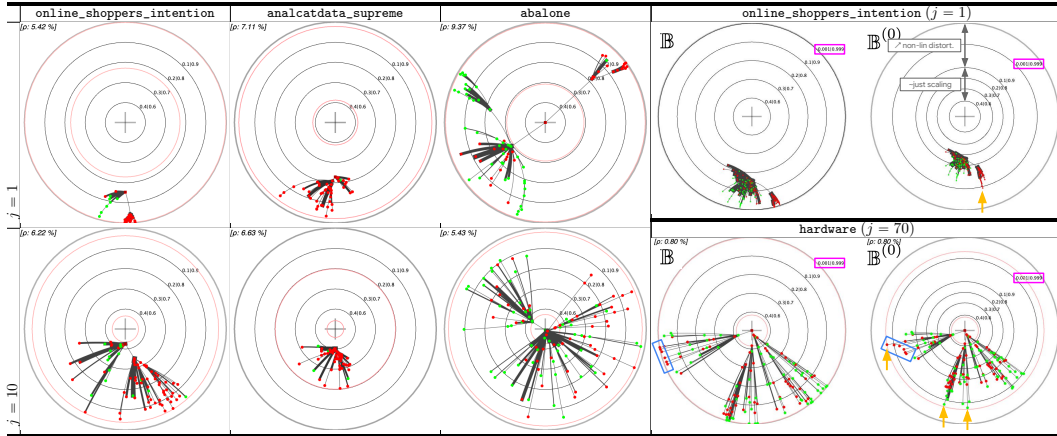

Table 1: *Left pane*: embedding in Poincaré disk of the MDTs corresponding to the DTs learned at the $1^{st}$ (top row) and $10^{th}$ (bottom row) boosting iteration, on four UCI domains. Stark differences emerge between these domains from the plots alone. *Right pane*: comparison between Poincaré disk embedding and its $t = 0$ t-self for an MDT learned on UCI `online_shoppers_intention` (top, boosting coefficients information not shown) and UCI `hardware` (bottom). The $p \in \{0.001, 0.999\}$ isoline (rectangle) is barely distinguishable from $\partial\mathbb{B}$ but is clearly distinct from $\partial\mathbb{B}^{(0)}$. Note that in $\mathbb{B}^{(0)}$, the center looks similar to a scaling (zoom) of $\mathbb{B}$; while near the border, the high nonlinearity of $\mathbb{B}^{(0)}$ allows us to spot nodes that have high confidence / training accuracy (in orange) but can hardly be distinguished from the bulk of "just good" nodes in $\mathbb{B}$. Note also the set of red nodes in the Poincaré disk for $j = 70$ (rectangle) that mistakenly look aligned, but not in the t-self.

All three conditions point to a non-linear embedding $\mathfrak{p}_t : \mathbb{B} \to \mathbb{B}^{(t)}$. The $t$-self offers a simple convenient solution with a trivial design for $\mathfrak{p}_t$: We compute $z^{(t)}$ by a simple scaling of the Euclidean norm of $z$ in $\mathbb{B}$ to ensure that (13) holds. Figure 3 (right) displays the corresponding relationship between $r \doteq \|z\|$ and $r^{(t)} \doteq \|z^{(t)}\|$ when $t \in [0, 1]$, clearly achieving (ii) in addition to (iii). For (i), even for $t < 1$ very close to 1 and $z$ close to $\partial\mathbb{B}$ ($r$ close to 1), the mapping can send $z^{(t)}$ substantially "back in" the t-self $\mathbb{B}^{(t)}$: for $t = 0.7$ and $r = 1 - 10^{-4}$ – *i.e.* $k = 4$ in Definition 5.1 – we get $r^{(t)} \approx 0.96$, authorizing encoding with a less "risky" $k = 2$. Two additional benefits from (13), visible from Figure 3 (right) is that the distortion is low near the center and then merely affine in a wide region until near the heavy non-linear changes, typically when $r > 0.8$.

**Remark 3.** *The Appendix details properties of the t-self of the Lorentz model of hyperbolic geometry, another popular model (Section C.V).*

# 6 Experiments

We summarize a number of experiments (Table 1), provided otherwise *in extenso* in the Appendix. In the top-down induction scheme for DT, the leaf chosen to be split is the heaviest non pure leaf, *i.e.* the one among the leaves containing both classes with the largest proportion of training examples. Induction is stopped when the tree reaches a fixed size, or when all leaves are pure, or when no further split allows decreasing the expected log-loss. We do not prune DTs. Our domains are public (*Cf* Appendix).

**Poincaré embeddings of DTs / MDTs** See Figure 2 (left) for a summary of the visualization. We remind that the center of the disk is "poor" classification (confidence 0), or, in the context of boosting, random classification. A striking observation is the sheer variety of patterns that are plainly obvious from our visualization but would otherwise be difficult to spot using classical tools. Some are common to all domains: as boosting iterations increase, low-depth tree nodes tend to converge to the center of the disk, indicating increased hardness in classification. This is the impact of a well known effect of boosting, whose weight modifications make the problem "harder". Among domain-dependent patterns, highly imbalanced domains get a root predictably initially embedded "far" from the origin (`online_shoppers_intention`, `analcatdata_supreme`) while more balanced domains get their roots embedded near the origin (`abalone`). Across the experiment, all trees have at most 200 nodes: while this clearly provides very good models after a large number of iterations on some domains, it is not enough to get good models after just a few iterations on others (`analcatdata_supreme`). Interpreting the hyperbolic embeddings can also be telling: consider Figure 2 (right), which a MDT of the ($j = 3$) boosted DT learned on `online_shoppers_intention`. Notice the low embedding error (5.46%). We see in (A) (magenta) that classes are more balanced at the root compared to previous trees (All DTs in Appendix, Table IV) because the root is closer to the center of the disk: thus, hard examples belong to both classes (no class becomes much harder to "learn" than the other). Two clearly distinct subtrees are associated to non-purchase patterns (red, A). The bottom-most subtree achieves very good confidence with several leaves close to the border: these are non-purchase patterns clearly distinct from purchase patterns (green nodes); the whole subtree dangles from the root via a test on feature PageValues (grey, B) achieving a substantial boost in confidence; many nodes are way past posterior isoline $p \in \{0.1, 0.9\}$. Red subtree in (A) is a lot closer to purchase patterns (C). One distinct pattern of purchase emerges, built from tests that always *strictly* increase its prediction's confidence (orange). The full rule achieves very high accuracy (leaf posterior nears 0.99).

**Interest of the t-self for visualization** As demonstrated by Table 1 (right) and Appendix, the t-self is particularly useful to tell the best parts of a tree. Because it also manages to "push back" the bulk of nodes from the border $\partial \mathbb{B}^{(t)}$, it displays the t-self might also be useful as a *standard* encoding (use (13) to access Poincaré disk quantities and embedding).

**MDT vs DT for classification** We traded the hyperbolic visualization of a DT – bound to be poor – for the visualization of its "monotonic part", an MDT. We have seen that the MDT visualization provides lots of clues about the DT as well. In terms of classification, the fact that the set of leaves of the MDT is a subset of the set of leaves of its DT hints on similarities in classification as well, but so far we have not explored a key question whose scope goes beyond this paper's: how does the classification of a MDT compares to its DT ? This question is important because in addition to being more amenable to visualization than DTs, it might also be the case that classification with MDTs might just be a good alternative to doing it with DTs. The scope of this question goes beyond this paper because of the tremendous popularity of DTs. The left pane of Table 2 provides a partial answer on a subset of our domains (the full set of results is in the Appendix, Section D.II; p-values are that of a Student paired t-test with H0 being the identity of the average errors). Clearly, the classification accuracies are similar for a majority of the domains considered, but interestingly, we observe the two polarities when they are not: on domains `ionosphere` and `online_shoppers_intention`, the DT is better than its MDT, but on domain `analcatdata_supreme` it is the opposite. This is particularly interesting because the MDT is no larger and can be substantially smaller than its corresponding DT and thus could represent an efficient pruning of its DT.

**MDT embedding: visualization error** The right pane of Table 2 provides an example of embedding errors $\rho$ (5) for the first five MDTs in a boosted ensemble (we remind that the description of the boosting algorithm and the embedding of the leveraging coefficients is provided in Appendix, Section C.IX). The full set of results is in Appendix, Section D.II. The results show that errors are small in general, which is good news given our quite heuristic modification to Sarkar's embedding. There is more: the embedding error tends to decrease – in some cases very significantly – with the MDT's index in the ensemble, see for example `winered`, `german` and `qsar`. The explanation is simple and

| domain | test error (DS vs MDT) | | | MDT # in ens. + visu. error $\rho$ | | | | |
|---|---|---|---|---|---|---|---|---|
| | DT | MDT | p-val | #1 | #2 | #3 | #4 | #5 |
| `ionosphere` | $5.71 \pm 2.70$ | $9.11 \pm 5.34$ | 0.0237 | 15.21 | 9.84 | 8.20 | 6.88 | 3.45 |
| `winered` | $18.39 \pm 2.02$ | $18.32 \pm 2.27$ | **0.9227** | 12.13 | 11.87 | 8.56 | 6.31 | 4.44 |
| `german` | $24.00 \pm 4.37$ | $23.90 \pm 4.25$ | **0.8793** | 14.74 | 10.06 | 7.73 | 5.05 | 2.70 |
| `a._s.` | $23.40 \pm 1.85$ | $22.56 \pm 1.75$ | 0.0134 | 7.11 | 8.32 | 8.23 | 8.67 | 7.60 |
| `abalone` | $22.15 \pm 2.20$ | $21.14 \pm 2.44$ | **0.1267** | 9.37 | 6.53 | 8.20 | 7.61 | 8.18 |
| `qsar` | $12.98 \pm 2.63$ | $12.89 \pm 4.05$ | **0.8772** | 16.78 | 12.97 | 12.86 | 5.77 | 4.24 |
| `o._s._i.` | $9.90 \pm 0.78$ | $10.67 \pm 0.54$ | 0.0057 | 5.42 | 3.74 | 5.46 | 6.42 | 7.00 |
| `hardware` | $1.33 \pm 0.27$ | $1.44 \pm 0.19$ | **0.0534** | 7.78 | 8.03 | 3.44 | 3.49 | 2.48 |

Table 2: Summary of two experiments for a subset of our domains. Left pane: test errors (%) of DTs vs their corresponding MDTs (average±std dev). **Bold faces** on p-values indicate *keeping* the identity of averages for a 0.05 first-order risk. Right pane: embedding error $\rho$ (5) (%) for the visualization of the first five MDTs in an ensemble learned. Shorthands used: `o._s._i.` = `online_shoppers_intention`; `a._s.` = `analcatdata_supreme`. See text for details.

due to a key boosting feature, the reweighting mechanism underlined above: with weight updates that bring in general the total (new) weight of positive examples closer to that of negative examples, the root of the next MDT gets embedded closer to the center of the disk. As the root moves closer to the center, it is much easier to get a good spread of the tree in the rest of the disk at reduced error, in particular if the MDT is well balanced. For an example, see `hardware` for $j = 70$ in Figure 1 ($\rho < 1\%$).

# 7   Conclusion

This paper proposes three separate contributions that altogether provide a solution for hyperbolic embedding of (ensembles of) decision trees (DT), via (i) a link between losses for class probability estimation and hyperbolic distances, (ii) the design of a clean, unambiguous embedding of a DT via its monotonic subtree and (iii) a way to improve visualization and encoding properties of the Poincaré disk model using its t-self, a notion we introduce. Each of these contributions are indeed separate: for example, one could reuse (i) to embed models different from DTs or (iii) to perform hyperbolic embeddings of objects being not even related to supervised models. Contribution (ii) is of independent interest with the introduction of a new kind of tree-based models, monotonic decision trees.

We believe that the way we address (iii) opens general applications even outside hyperbolic geometry. Indeed, we generalize classical integration and derivation to a context encompassing the concept of additivity, upon which integration is built, extending standard properties of integration and derivation in a natural way (see the Appendix for many examples). That such properties can be obtained without additional technical "effort" bodes well for perspectives of developments and applications in other subfields of ML, where such tools could be used, not just in our geometric context, to smoothly tweak distortion measures. Many distortion measures used in ML are indeed integrals in nature: Bregman divergences, $f$-divergences, integral probability metrics, etc.
One inherent limitation of our work is linked to the use of DTs: they typically fit very well to tabular variables (attribute-value data) but otherwise should be used with caution. Our work is a first step towards more sophisticated visualization for supervised models (Section 2), looking for tailored fit geometric spaces to best blend their symbolic and numerical properties. More can be done and more has to be done: at an age of data collection and ML compute still ramping up, seeking model "pictures" worth a thousand words is a challenge but a necessity for responsible AI and explainability. Finally, our introduction of monotonic decision trees begs for a deeper statistical analysis of such models, given the tremendous popularity of decision trees that they could complete – or replace – on mainstream data science pipelines.

## Acknowledgments

Thanks are due to Mathieu Guillame-Bert, reviewers and the area chair for many suggestions that helped to improve the paper's content. Work was done while AS was visiting RIKEN AIP.

# References

[1] E. Amid, F. Nielsen, R. Nock, and M.-K. Warmuth. Optimal transport with tempered exponential measures. In *AAAI'24*, 2024.

[2] Ehsan Amid, Frank Nielsen, Richard Nock, and Manfred K Warmuth. The tempered Hilbert simplex distance and its application to non-linear embeddings of TEMs. *arXiv preprint arXiv:2311.13459*, 2023.

[3] A. Banerjee, S. Merugu, I. Dhillon, and J. Ghosh. Clustering with Bregman divergences. In *Proc. of the $4^{th}$ SIAM International Conference on Data Mining*, pages 234–245, 2004.

[4] A.-F. Beardon and D. Minda. The hyperbolic metric and geometric function theory. In *Proc. of the International Workshop on Quasiconformal Mappings and their Applications*, 2005.

[5] B.-H. Bowditch. *A course on geometric group theory*. Math. Soc. Japan Memoirs, 2006.

[6] I. Chami, A. Gu, V. Chatziafratis, and C. Ré. From trees to continuous embeddings and back: Hyperbolic hierarchical clustering. In *NeurIPS*33*, 2020.

[7] I. Chami, A. Gu, D. Nguyen, and C. Ré. Horopca: Hyperbolic dimensionality reduction via horospherical projections. In $38^{th}$ *ICML*, volume 139 of *Proceedings of Machine Learning Research*, pages 1419–1429. PMLR, 2021.

[8] P. Chlenski, E. Turok, A. Khalil Moretti, and I. Pe'er. Fast hyperboloid decision tree algorithms. In $12^{th}$ *ICLR*, 2024.

[9] H. Cho, B. Demeo, J. Peng, and B. Berger. Large-margin classification in hyperbolic space. In $22^{nd}$ *AISTATS*, volume 89 of *Proceedings of Machine Learning Research*, pages 1832–1840. PMLR, 2019.

[10] Z. Cranko and R. Nock. Boosted density estimation remastered. In $36^{th}$ *ICML*, pages 1416–1425, 2019.

[11] L. Doorenbos, P. Márquez-Neila, R. Sznitman, and P. Mettes. Hyperbolic random forests. *CoRR*, abs/2308.13279, 2023.

[12] D. Dua and C. Graff. UCI machine learning repository, 2021.

[13] J. Friedman, T. Hastie, and R. Tibshirani. Additive Logistic Regression : a Statistical View of Boosting. *Ann. of Stat.*, 28:337–374, 2000.

[14] O.-E. Ganea, G. Bécigneul, and T. Hofmann. Hyperbolic entailment cones for learning hierarchical embeddings. In $35^{th}$ *ICML*, volume 80 of *Proceedings of Machine Learning Research*, pages 1632–1641. PMLR, 2018.

[15] O.-E. Ganea, G. Bécigneul, and T. Hofmann. Hyperbolic neural networks. In *NeurIPS*31*, pages 5350–5360, 2018.

[16] D.-E. Knuth. Two notes on notation. *The American Mathematical Monthly*, 99(5):403–422, 1992.

[17] M.-T. Law, R. Liao, J. Snell, and R.-S. Zemel. Lorentzian distance learning for hyperbolic representations. In $36^{th}$ *ICML*, volume 97 of *Proceedings of Machine Learning Research*, pages 3672–3681. PMLR, 2019.

[18] Y. Mansour, R. Nock, and R.-C. Williamson. Random classification noise does not defeat all convex potential boosters irrespective of model choice. In $40^{th}$ *ICML*, 2023.

[19] G. Mishne, Z. Wan, Y. Wang, and S. Yang. The numerical stability of hyperbolic representation learning. In $40^{th}$ *ICML*, 2023.

[20] M. Nickel and D. Kiela. Poincaré embeddings for learning hierarchical representations. In *NeurIPS*30*, pages 6338–6347, 2017.

[21] F. Nielsen and R. Nock. On Rényi and Tsallis entropies and divergences for exponential families. *CoRR*, abs/1105.3259, 2011.

[22] L. Nivanen, A. Le Méhauté, and Q.-A. Wang. Generalized algebra within a nonextensive statistics. *Reports on Mathematical Physics*, 52:437–444, 2003.

[23] R. Nock, W. Bel Haj Ali, R. D'Ambrosio, F. Nielsen, and M. Barlaud. Gentle nearest neighbors boosting over proper scoring rules. *IEEE Trans.PAMI*, 37(1):80–93, 2015.

[24] R. Nock and A. K. Menon. Supervised learning: No loss no cry. In $37^{th}$ *ICML*, 2020.

[25] M.-C. Pardo and I. Vajda. About distances of discrete distributions satisfying the data processing Theorem of Information Theory. *IEEE Trans. IT*, 43:1288–1293, 1997.

[26] J. R. Quinlan. *C4.5 : programs for machine learning*. Morgan Kaufmann, 1993.

[27] J.-G. Ratcliffe. *Foundations of Hyperbolic Manifolds*. Springer Graduate Texts in Mathematics, 1994.

[28] M.-D. Reid and R.-C. Williamson. Information, divergence and risk for binary experiments. *JMLR*, 12:731–817, 2011.

[29] F. Sala, C. De Sa, A. Gu, and C.Ré. Representation tradeoffs for hyperbolic embeddings. In $35^{th}$ *ICML*, volume 80 of *Proceedings of Machine Learning Research*, pages 4457–4466. PMLR, 2018.

[30] R. Sarkar. Low distortion Delaunay embedding of trees in hyperbolic plane. In *GD'11*, volume 7034, pages 355–366. Springer, 2011.

[31] R. E. Schapire and Y. Singer. Improved boosting algorithms using confidence-rated predictions. *MLJ*, 37:297–336, 1999.

[32] Andrew D Selbst and Solon Barocas. The intuitive appeal of explainable machines. *Fordham L. Rev.*, 87:1085, 2018.

[33] A. Slavik. *Product integration, its history and applications*. Matfyzpress, Praze, 2007.

[34] R. Sonthalia and A.-C. Gilbert. Tree! I am no tree! I am a low dimensional hyperbolic embedding. In *NeurIPS*33*, 2020.

[35] T. van Erven and P. Harremoës. Rényi divergence and kullback-leibler divergence. *IEEE Trans. IT*, 60:3797–3820, 2014.

[36] M. Yang, M. Zhou, R. Ying, Y. Chen, and I. King. Hyperbolic representation learning: Revisiting and advancing. In $40^{th}$ *ICML*, volume 202 of *Proceedings of Machine Learning Research*, pages 39639–39659, 2023.

[37] T. Yu and C. De Sa. Numerically accurate hyperbolic embeddings using tiling-based models. In *NeurIPS*32*, pages 2021–2031, 2019.

[38] T. Yu, T.-J.-B. Liu, A. Tseng, and C. De Sa. Shadow cones: Unveiling partial orders in hyperbolic space. *CoRR*, abs/2305.15215, 2023.

# Appendix

To differentiate with the numberings in the main file, the numbering of Theorems, etc. is letter-based (A, B, ...).

## Table of contents

## A  Broader Impact

This paper presents work whose goal is to advance the field of Machine Learning. Beyond our proposed tempered calculus which is used to amend the original Poincaré disk model of hyperbolic geometry, the monotonic decision trees and $t$-self hyperbolic embedding contribute to the field of visualization and explainable AI for ML models. Thus, there are two distinct avenues for broader impact and related care:

1. from the geometry standpoint, the $t$-self brings a novel coding and visualization advantage over the founding Poincaré disk model for the embedding close to the disk's border, regardless of the nature of the embedding (whether it is data, models, etc. that are embedded). However, being new, it is important to keep the label that the $t$-self is being used, and also clearly display the relevant value of $t$ or ranges, since the properties of the embedding can change depending on them;

2. from the model standpoint and a societal perspective, these visualizations will provide better methods for deployed decision tree models to be scrutinized. A potential negative of our approach is the required reduction from normal decision trees to monotonic decision trees, where the visualization does not directly correspond to the initial decision tree. We believe that monotonicity adds value to the interpretation of rules that are progressively built from trees, yet practitioners should be careful when extracting inferences from the reduced monotonic decision tree and clearly label them as being extracted from such a model, to prevent confusion with decision trees, that are overwhelmingly popular. From a pure prediction performance standpoint, we show – at least empirically within the bounds of our settings – that monotonic decision trees are similar to their original decision tree counterparts.

## B  $t$-algebra and $t$-additivity

We provide here a few more details on the basis of our paper, the $t$-algebra and the $t$-additivity of some divergences.

### B.I  Primer on $t$-algebra

Classical arithmetic over the reals can be used to display duality relationships between operators using the $\log, \exp$ functions, such as for example $\log(a/b) = \log a - \log b$, $\exp(a + b) = \exp(a) \cdot \exp(b)$, and so on. They can also be used to define one operator from another one. There is no difference between the operators appearing inside and outside functions. In the $t$-algebra, a difference appears and such relationships can be used to define the $t$-operators from those over the reals, as indeed one can define the tempered addition

$$x \oplus_t y \quad \doteq \quad \log_t(\exp_t(x) \cdot \exp_t(y)),$$

the tempered subtraction,

$$x \ominus_t y \quad \doteq \quad \log_t(\exp_t(x)/\exp_t(y)),$$

(both simplifying to the expressions we use), and of course the tempered product and division,

$$x \otimes_t y \doteq \exp_t(\log_t(x) + \log_t(y)) \quad ; \quad x \oslash_t y \doteq \exp_t(\log_t(x) - \log_t(y)),$$

whose simplified expression appears *e.g.* in [1]. See also [22].

### B.II  Functional $t$-additivity

As is well-known, Boltzman-Gibbs (and so, Shannon) entropy is additive while Tsallis entropy is $t$-additive. Note that additivity for BG requires being on the simplex, but $t$-additivity of Tsallis technically requires only positive measures – the simplex restriction ensures the limit exists for $t \to 1$ and then T→BG.

**Divergences can also be $t$-additive** The KL divergence

$$D_{\mathrm{KL}}(\boldsymbol{p}\|\boldsymbol{q}) \quad \doteq \quad \sum_k p_k \log(p_k/q_k)$$

is additive on the simplex but not $t$-additive: using a decomposition of $\boldsymbol{p}, \boldsymbol{q}$ as product of (independent) distributions ($\boldsymbol{p}_1, \boldsymbol{p}_2$ and $\boldsymbol{q}_1, \boldsymbol{q}_2$) using the cartesian product of their support, we indeed check $D_{\mathrm{KL}}(\boldsymbol{p}\|\boldsymbol{q}) = D_{\mathrm{KL}}(\boldsymbol{p}_1\|\boldsymbol{q}_1) + D_{\mathrm{KL}}(\boldsymbol{p}_2\|\boldsymbol{q}_2)$ with $p_{ij} = p_{1i}p_{2j}$ and $q_{ij} = q_{1i}q_{2j}$. On the other hand, Tsallis divergence [21] is $t$-additive on positive measures with such a decomposition, with

$$D_{\mathrm{T}}(\boldsymbol{p}\|\boldsymbol{q}) \doteq \frac{\sum_k p_k(p_k/q_k)^{1-t} - 1}{1-t},$$

and we check that $D_{\mathrm{T}}(\boldsymbol{p}\|\boldsymbol{q}) = D_{\mathrm{T}}(\boldsymbol{p}_1\|\boldsymbol{q}_1) + D_{\mathrm{T}}(\boldsymbol{p}_2\|\boldsymbol{q}_2) + (1-t) \cdot D_{\mathrm{T}}(\boldsymbol{p}_1\|\boldsymbol{q}_1) \cdot D_{\mathrm{T}}(\boldsymbol{p}_2\|\boldsymbol{q}_2)$ (additional requirement to be on the simplex for convergence to $D_{\mathrm{KL}}$ as $t \to 1$). For $t \notin (1,2)$ and on the simplex, Tsallis divergence is an $f$-divergence with generator $z \mapsto (z^{2-t} - 1)/(1-t)$. Similarly, the tempered relative entropy is $t$-additive *on the co-simplex*, where in this case

$$D_t(\boldsymbol{p}\|\boldsymbol{q}) \doteq \frac{1 - \sum_k p_k q_k^{1-t}}{1-t}, \boldsymbol{p}, \boldsymbol{q} \in \tilde{\Delta}_m.$$

The tempered relative entropy is a Bregman divergence with generator $z \mapsto z \log_t z - \log_{t-1} z$.

## C   Supplementary material on proofs

### C.I   Proof of Theorem 2

The Theorem is tautological for $t = 1$ so we prove it for $t \neq 1$. Denote $\mathcal{P}(\mathcal{S})$ the set of subsets of set $\mathcal{S}$ and $\mathcal{P}_*(\mathcal{S}) \doteq \mathcal{P}(\mathcal{S}) \backslash \{\varnothing\}$. We first transform $S^{(t)}_{\Delta_n}(f)$ (index $n$ shown for readability) in a better suited expression:

$$
\begin{aligned}
S^{(t)}_{\Delta_n}(f) &\doteq (\bigoplus_t)_{i=1}^n |\mathbb{I}_i| f(\xi_i) \\
&= \sum_{P \in \mathcal{P}_*([n])} (1-t)^{|P|-1} \cdot \prod_{i \in P} |\mathbb{I}_i| f(\xi_i) \\
&= \frac{1}{1-t} \cdot \sum_{P \in \mathcal{P}_*([n])} \prod_{i \in P} (1-t)|\mathbb{I}_i| f(\xi_i) \\
&= \frac{1}{1-t} \cdot \left( \prod_{i=1}^n (1 + (1-t)|\mathbb{I}_i| f(\xi_i)) - 1 \right),
\end{aligned}
\tag{14}
$$

where we have used $\mathbb{I}_i \doteq [x_{i-1}, x_i]$ for conciseness. We now need a technical Lemma.

**Lemma 2.** *Fix any $0 \leqslant v < 8/10$ and consider any $n$ reals $q_i, i \in [n]$ such that $|q_i| \leqslant v, \forall i \in [n]$. Then*

$$1 \leqslant \frac{\left(1 + \frac{1}{n} \cdot \sum_i q_i\right)^n}{\prod_i (1 + q_i)} \leqslant \exp(nv \cdot v). \tag{15}$$

*Proof.* Suppose all the $q_i, i \in [n]$ satisfy $q_i \in [u, v]$ and let $\varphi$ be strictly convex differentiable and defined over $[u, v]$. Then it comes from [10, Lemma 9] that we get the right-hand side of

$$0 \leqslant \mathbb{E}_i[\varphi(q_i)] - \varphi(\mathbb{E}_i[q_i]) \leqslant D_\varphi\left(w \,\middle\|\, (\varphi')^{-1}\left(\frac{\varphi(u) - \varphi(v)}{u - v}\right)\right), \tag{16}$$

where we can pick $w \in \{u, v\}$ and $D_\varphi$ is the Bregman divergence with generator $\varphi$ (the left-hand side is Jensen's inequality). Picking $u \doteq -v$ (assuming wlog $v > 0$) and letting

$$Y_v \doteq \frac{1-v}{2v} \cdot \log\left(1 + \frac{2v}{1-v}\right), \tag{17}$$

for the choice $\varphi(z) \doteq -\log(1 + z)$ and $w \doteq -v$, we obtain after simplification

$$0 \leqslant \log\left(1 + \frac{1}{n} \cdot \sum_i q_i\right) - \frac{1}{n} \cdot \sum_i \log(1 + q_i) \leqslant Y_v - \log(Y_v) - 1. \tag{18}$$

Analizing function $v \mapsto Y_v - \log(Y_v) - 1$ reveals that it is upperbounded by $v \mapsto v^2$ if $v \in [-8/10, 8/10]$. Hence, multiplying by $n$ all sides and passing to the exponential, we get that

$$1 \leqslant \frac{\left(1 + \frac{1}{n} \cdot \sum_i q_i\right)^n}{\prod_i (1 + q_i)} \leqslant \exp\left(nv^2\right), \forall 0 \leqslant v < \frac{8}{10},$$

which leads to the statement of the Lemma. □

Let us come back to our Riemannian summation setting and let

$$
\begin{aligned}
q_i &\doteq (1 - t) \cdot |\mathbb{I}_i| f(\xi_i), \forall i \in [n], \\
v &\doteq \max_i |q_i|.
\end{aligned}
$$

Assume now that $f$ is Riemann integrable, which allows to guarantee that, at least for $n$ large enough and a step of division $\Delta_n$ not too large, we have $|q_i| \leqslant 8/10, \forall i \in [n]$ and so we can use Lemma 2. We get from (14) and Lemma 2, if $t < 1$

$$S_{\Delta_n}^{(t)}(f)$$

$$= \frac{1}{1-t} \cdot \left(\prod_{i=1}^n (1 + q_i) - 1\right)$$

$$\in \left[\frac{1}{1-t} \cdot \left(\left(1 + \frac{1}{n} \cdot \sum_i q_i\right)^n \cdot \exp(-nv \cdot v) - 1\right), \frac{1}{1-t} \cdot \left(\left(1 + \frac{1}{n} \cdot \sum_i q_i\right)^n - 1\right)\right] \quad (19)$$

and we permute the bounds if $t > 1$. Importantly, we note that

$$\sum_i q_i = (1 - t) \cdot S_{\Delta_n}^{(1)}(f). \quad (20)$$

So suppose $\lim_{n \to +\infty} S_{\Delta_n}^{(1)}(f) \doteq L_1 \doteq \int_a^b f(u) u$ is finite and choose the step of division $\Delta_n$ not too large so that

$$n \cdot \max_i |q_i| = (1 - t) \cdot \max_i(n|\mathbb{I}_i|)|f(\xi_i)| \leqslant K \cdot ((1-t)|b - a| \max f([a,b])),$$

for some constant $K \geqslant 1$ (this is possible because $f$ is (1-)Riemann integrable; we also note that if the division is regular then we can choose $K = 1$). The value of $K$ is not important: what is important is that $nv$ remains finite in (19) as $n$ increases, while $\lim_{n \to +\infty} v = 0$. Hence, $\exp(-nv \cdot v) \to 1$ as $n \to +\infty$ and (19) implies, because $\lim_{n \to +\infty}(1 + a/n)^n = \exp a$, the two first identities in

$$
\begin{aligned}
\lim_{n \to +\infty} S_{\Delta_n}^{(t)}(f) &= \frac{1}{1-t} \cdot \left(\lim_{n \to +\infty}\left(1 + \frac{1}{n} \cdot \sum_i q_i\right)^n - 1\right) \\
&= \frac{1}{1-t} \cdot \left(\exp\left(\sum_i q_i\right) - 1\right) \\
&= \frac{1}{1-t} \cdot (\exp((1-t)L_1) - 1) \quad (21) \\
&= \log_t \exp(L_1) \\
&= \log_t \exp\left(\int_a^b f(u)\mathrm{d}u\right)
\end{aligned}
$$

((21) follows from (20)) and by definition $\lim_{n \to +\infty} S_{\Delta_n}^{(t)}(f) \doteq \overset{(t)}{\int_a^b} f(u)\mathrm{d}_t u$. So we get that Riemann integration ($t = 1$) grants

$$\overset{(t)}{\int_a^b} f(u)\mathrm{d}_t u = \log_t \exp \int_a^b f(u)\mathrm{d}u,$$

which, in addition to showing (9) (main file) also shows that $t = 1$-Riemann integration is equivalent to all $t \neq 1$-Riemann integration, ending the proof of Theorem 2.

**Remark 4.** *The absence of affine terms in upperbounding $v \mapsto Y_v - \log(Y_v) - 1$ in the neighborhood of 0 is crucial to get to our result.*

## C.II  Proof of Theorem 3

Using Theorem 2, we just have to analyze the limit in relationship to the Riemannian case:

$$
\begin{aligned}
\mathrm{D}_t F(z) &\doteq \lim_{\delta \to 0} \frac{\overset{(t)}{\int_a^{z+\delta}} f(u)\mathrm{d}_t u \ominus_t \overset{(t)}{\int_a^{z}} f(u)\mathrm{d}_t u}{\delta} \\
&= \lim_{\delta \to 0} \frac{\log_t \exp \int_a^{z+\delta} f(u)\mathrm{d}u \ominus_t \log_t \exp \int_a^z f(u)\mathrm{d}u}{\delta} \quad (22) \\
&= \lim_{\delta \to 0} \frac{1}{\delta} \cdot \log_t \left( \frac{\exp \int_a^{z+\delta} f(u)\mathrm{d}u}{\exp \int_a^z f(u)\mathrm{d}u} \right) \quad (23) \\
&= \lim_{\delta \to 0} \frac{1}{\delta} \cdot \log_t \exp \left( \int_a^{z+\delta} f(u)\mathrm{d}u - \int_a^z f(u)\mathrm{d}u \right) \\
&= \lim_{\delta \to 0} \frac{1}{\delta} \cdot \left( \int_a^{z+\delta} f(u)\mathrm{d}u - \int_a^z f(u)\mathrm{d}u \right) \quad (24) \\
&\doteq \mathrm{D}_1 F(z) = f,
\end{aligned}
$$

where the last identity is the classical Riemannian case, (22) follows from Theorem 2, (23) follows from a property of $\log_t$ ($\log_t a \ominus \log_t b = \log_t(a/b)$), (24) follows from the fact that $\log_t \exp(z) =_0 z + o(z)$. This completes the proof of Theorem 3.

## C.III  Additional and helper results for Theorems 2 and 3

We first state a trivial but important Lemma, whose content is partially used in the main file.

**Lemma 3.** $\mathfrak{g}_t$ *satisfies the following properties:*

1. *$\mathfrak{g}_t(z)$ is strictly increasing for any $t \in \mathbb{R}$, strictly concave for any $t > 1$, strictly convex for any $t < 1$ and such that $\mathrm{sign}(\mathfrak{g}_t(z)) = \mathrm{sign}(z), \forall t \in \mathbb{R}$;*
2. *$\mathrm{D}_t \mathfrak{g}_t(z) = 1$;*
3. *(t-integral mean-value) Suppose $f$ Riemann integrable over $[a,b]$. Then there exists $c \in (a,b)$ such that*

$$
(b-a) \cdot \mathfrak{g}_{t'} \circ f(c) = \overset{(t)}{\int_a^b} f(u)\mathrm{d}_t u \ \ (t' \doteq 1 - (1-t)(b-a)).
$$

We list a series of consequences to both theorems.

**General properties of $t$-integrals**  Some properties generalize those for classical Riemann integration.

**Theorem C.1.** *The following relationships hold for any $t \in \mathbb{R}$ and any functions $f, g$ $t$-Riemann integrable over some interval $[a,b]$:*

$$
\begin{aligned}
\overset{(t)}{\int_a^b} f(u)\{+ \text{ or } -\} g(u)\mathrm{d}_t u &= \left( \overset{(t)}{\int_a^b} f(u)\mathrm{d}_t u \right) \{\oplus_t \text{ or } \ominus_t\} \left( \overset{(t)}{\int_a^b} g(u)\mathrm{d}_t u \right) & (\textit{additivity}), \\
\overset{(t)}{\int_a^b} \lambda f(u)\mathrm{d}_t u &= \lambda \cdot \overset{(1-(1-t)\lambda)}{\int_a^b} f(u)\mathrm{d}_t u \quad (\lambda \in \mathbb{R}) & (\textit{dilativity}), \\
\overset{(t)}{\int_a^b} f(x)\mathrm{d}_t u &= \overset{(t)}{\int_a^c} f(u)\mathrm{d}_t u \oplus_t \overset{(t)}{\int_c^b} f(u)\mathrm{d}_t u \quad (c \in [a,b]) & (\textit{Chasles' relationship}), \\
\left| \overset{(t)}{\int_a^b} f(u)\mathrm{d}_t u \right| &\leqslant \overset{(1-|1-t|)}{\int_a^b} |f(u)|\mathrm{d}_t u & (\textit{triangle inequality}), \\
\overset{(t)}{\int_a^b} f(u)\mathrm{d}_t u &\leqslant \overset{(t)}{\int_a^b} g(u)\mathrm{d}_t u \quad (f \leqslant g) & (\textit{monotonicity}).
\end{aligned}
$$

*Proof.* We show additivity for $\oplus_t/+$ (the same path shows the result for $\ominus_t/-$):

$$
\begin{aligned}
\overset{(t)}{\int_a^b} f(u)\mathrm{d}_t u \oplus_t \overset{(t)}{\int_a^b} g(u)\mathrm{d}_t u &= \log_t \exp \int_a^b f(u)\mathrm{d}u \oplus_t \log_t \exp \int_a^b g(u)\mathrm{d}u \\
&= \log_t \left( \exp \int_a^b f(u)\mathrm{d}u \cdot \exp \int_a^b g(u)\mathrm{d}u \right) \\
&= \log_t \exp \int_a^b (f+g)(u)\mathrm{d}u \\
&= \overset{(t)}{\int_a^b} (f+g)(u)\mathrm{d}_t u.
\end{aligned}
$$

We show dilativity, using $t' \doteq 1 - (1-t)\lambda$:

$$
\begin{aligned}
\overset{(t)}{\int_a^b} \lambda f(u)\mathrm{d}_t u &= \log_t \exp \int_a^b \lambda f(u)\mathrm{d}u \\
&= \log_t \exp \lambda \int_a^b f(u)\mathrm{d}u \\
&= \frac{1}{1-t} \cdot \left( \exp\left( \lambda(1-t) \int_a^b f(u)\mathrm{d}u \right) - 1 \right) \\
&= \frac{1-t'}{1-t} \cdot \frac{1}{1-t'} \cdot \left( \exp\left( (1-t') \int_a^b f(u)\mathrm{d}u \right) - 1 \right) \\
&= \lambda \cdot \frac{1}{1-t'} \cdot \left( \exp\left( (1-t') \int_a^b f(u)\mathrm{d}u \right) - 1 \right) \\
&= \lambda \cdot \log_{t'} \exp \int_a^b f(u)\mathrm{d}u \\
&= \lambda \cdot \overset{(t')}{\int_a^b} f(u)\mathrm{d}_t u = \lambda \cdot \overset{(1-(1-t)\lambda)}{\int_a^b} f(u)\mathrm{d}_t u.
\end{aligned}
$$

The triangle inequality follows from the relationship:

$$
|\log_t \exp(z)| \quad \leqslant \quad \log_{1-|1-t|} \exp |z|, \forall z, t \in \mathbb{R},
$$

from which we use the fact that Riemann integration satisfies the triangle inequality and $\log_{1-|1-t|}$ is monotonically increasing on $\mathbb{R}_+$ in the penultimate line of:

$$
\begin{aligned}
\left| \overset{(t)}{\int_a^b} f(u)\mathrm{d}_t u \right| &\doteq \left| \log_t \exp \int_a^b f(u)\mathrm{d}u \right| \\
&\leqslant \log_{1-|1-t|} \exp \left| \int_a^b f(u)\mathrm{d}u \right| \\
&\leqslant \log_{1-|1-t|} \exp \int_a^b |f(u)|\mathrm{d}u \\
&= \overset{(1-|1-t|)}{\int_a^b} |f(u)|\mathrm{d}_t u.
\end{aligned}
$$

We show Chasles' relationship:

$$\overset{(t)}{\int_a^c} f(u)\mathrm{d}_t u \oplus_t \overset{(t)}{\int_c^b} f(u)\mathrm{d}_t u \;\doteq\; \log_t \exp \int_a^c f(u)\mathrm{d}u \oplus_t \log_t \exp \int_c^b f(u)\mathrm{d}u$$

$$= \; \log_t \left( \exp \int_a^c f(u)\mathrm{d}u \cdot \exp \int_c^b f(u)\mathrm{d}u \right)$$

$$= \; \log_t \exp \left( \int_a^c f(u)\mathrm{d}u + \int_c^b f(u)\mathrm{d}u \right)$$

$$= \; \log_t \exp \int_a^b f(u)\mathrm{d}u$$

$$= \; \overset{(t)}{\int_a^b} f(u)\mathrm{d}_t u,$$

where the second identity uses the property $\log_t a \oplus \log_t b = \log_t(ab)$. Monotonicity follows immediately from the fact that $z \mapsto \log_t \exp z$ is strictly increasing:

$$\overset{(t)}{\int_a^b} f(u)\mathrm{d}_t u \;=\; \log_t \exp \int_a^b f(u)\mathrm{d}u$$

$$\leqslant \; \log_t \exp \int_a^b g(u)\mathrm{d}u \doteq \overset{(t)}{\int_a^b} g(u)\mathrm{d}_t u.$$

This ends the proof of Theorem C.1. □

**Computing $t$-integrals**  Next, classical relationships to compute integrals do generalize to $t$-integration. We cite the case of integration by part.

**Lemma 4.** *Integration by part translates to $t$-integration by part as:*

$$\overset{(t)}{\int_a^b} fg'\mathrm{d}u \;=\; {}^{(t)}\left[ fg \right]_a^b \ominus_t \overset{(t)}{\int_a^b} f'g\mathrm{d}u,$$

*where we let*

$${}^{(t)}\left[ h \right]_a^b \;\doteq\; \log_t \exp(h(b)) \ominus_t \log_t \exp(h(a)).$$

(Proof immediate from Theorem 2)

**Geometric properties based on $t$-integrals**  This is a more specific result, important in the context of hyperbolic geometry: the well-known Hyperbolic Pythagorean Theorem (HPT) does translate to a tempered version with the same relationship to the Euclidean theorem. Consider a hyperbolic right triangle with hyperbolic lengths $a, b, c$, $c$ being the hyperbolic length of the hypothenuse. Let $a_t, b_t, c_t$ denote the corresponding tempered lengths, which are therefore explicitly related using $\mathfrak{g}_t$ as

$$a_t = \log_t \exp a, \quad b_t = \log_t \exp b, \quad c_t = \log_t \exp c.$$

Define the tempered generalization of $\cosh$:

$$\cosh_t z \;\doteq\; \frac{\exp_t z + \exp_t(-z)}{2}. \tag{25}$$

The HPT tells us that $\cosh c = \cosh a \cosh b$. It is a simple matter of plugging $\mathfrak{g}_t$, using the fact that $\log_t$ and $\exp_t$ are inverse of each other  and simplifying to get the tempered HPT, which we call $t$-HPT for short.

**Lemma 5.** *($t$-HPT) For any hyperbolic triangle described above, the tempered lengths are related as*

$$\cosh_t c_t = \cosh_t a_t \cosh_t b_t.$$

Now, remark that *for any* $t \neq 0$, a series expansion around 0 gives

$$\exp_t(z) = 1 + \frac{tz^2}{2} + o(z^3)$$

($\exp_t$ is always infinitely differentiable around 0, for any $t$) So the $t$-HPT gives

$$1 + \frac{tc_t^2}{2} + o(c_t^3) = \left(1 + \frac{ta_t^2}{2} + o(a_t^3)\right) \cdot \left(1 + \frac{tb_t^2}{2} + o(b_t^3)\right),$$

which simplifies, if we multiply both sides by $2/t$ and simplify it into

$$c_t^2 + o(c_t^3) = a_t^2 + b_t^2 + o(a_t^3) + o(b_t^3),$$

which for an infinitesimal right triangle gives $c_t^2 \approx a_t^2 + b_t^2$, *i.e.* Pythagoras Theorem, as does the HPT one gives in this case ($c^2 \approx a^2 + b^2$), which is equivalent of the particular $t = 1$-HPT case.

$t$-**mean value Theorem** The $t$-derivative yields a generalization of the Euclidean mean-value theorem.

**Lemma 6.** *Let* $t \in \mathbb{R}$ *and* $f$ *be continuous over an interval* $[a, b]$, *differentiable on* $(a, b)$ *and such that* $-1/(1-t) \notin f([a, b])$. *Then* $\exists c \in (a, b)$ *such that*

$$\mathrm{D}_t f(c) = \frac{(f(b) \ominus_t f(c)) - (f(a) \ominus_t f(c))}{b - a}.$$

*Proof.* We can obtain a direct expression of $\mathrm{D}_t f$ by using the definition of $\ominus_t$ and the classical derivative:

$$\mathrm{D}_t f(x) = \lim_{\delta \to 0} \frac{1}{\delta} \cdot \frac{f(x+\delta) - f(x)}{1 + (1-t)f(x)} = \frac{1}{1+(1-t)f(x)} \cdot \lim_{\delta \to 0} \frac{f(x+\delta) - f(x)}{\delta} = \frac{f'(x)}{1 + (1-t)f(x)}.$$
(26)

From here, the mean-value theorem tells us that there exists $c \in [a, b]$ such that

$$f'(c) = \frac{f(b) - f(a)}{b - a}$$

Dividing by $1 + (1-t)f(c)$ (assuming $f(c) \neq -1/(1-t)$) and reorganising, we get

$$\begin{aligned}
\mathrm{D}_t f(c) &= \frac{1}{b-a} \cdot \frac{f(b) - f(a)}{1 + (1-t)f(c)} \\
&= \frac{1}{b-a} \cdot \frac{f(b) - f(c)}{1 + (1-t)f(c)} - \frac{1}{b-a} \cdot \frac{f(a) - f(c)}{1 + (1-t)f(c)} \\
&= \frac{(f(b) \ominus_t f(c)) - (f(a) \ominus_t f(c))}{b - a},
\end{aligned}$$

which completes the proof of the Lemma. $\square$

In fact, the $t$-derivative of $f$ at some $c$ is "just" an Euclidean derivative for an affine transformation of the function, namely $z \mapsto f(z) \ominus_t f(c)$, also taken at $z = c$. This "proximity" between $t$-derivation and derivation is found in the tempered chain rule (proof straightforward).

**Lemma 7.** *Suppose* $g$ *differentiable at* $z$ *and* $f$ *differentiable at* $g(z)$. *Then*

$$\mathrm{D}_t(f \circ g)(z) = \mathrm{D}_t(f)(g(z)) \cdot g'(z).$$

## C.IV Sets endowed with a distortion and their $t$-self: statistical information

Here, $\mathcal{X}$ contains probability distributions or the parameters of probability distributions: $f$ can then be an $f$-divergence (information theory) or a Bregman divergence (information geometry). Tsallis divergence and the tempered relative entropy are examples of $t$-additive information theoretic and information geometric divergences. A key property of information theory is the data processing inequality (**D**), $\mathcal{X}$ being a probability space, which says that passing random variables through a Markov chain cannot increase their divergence as quantified by $f$ [25, 35]. A key property of

information geometry is the population minimizer property **(P)**, which elicits a particular function of a set of points as the minimizer of the expected distortion to the set, as quantified by $f$ [3]. We let **(J)** denote the joint convexity property, which would state for $f$ and any $\boldsymbol{x}_1, \boldsymbol{x}_2, \boldsymbol{y}_1, \boldsymbol{y}_2$ that

$$f(\lambda \cdot \boldsymbol{x}_1 + (1-\lambda) \cdot \boldsymbol{x}_2, \lambda \cdot \boldsymbol{y}_1 + (1-\lambda) \cdot \boldsymbol{y}_2) \leqslant \lambda f(\boldsymbol{x}_1, \boldsymbol{y}_1) + (1-\lambda) f(\boldsymbol{x}_2, \boldsymbol{y}_2), \qquad (27)$$

and convexity **(C)**, which amounts to picking $\boldsymbol{y}_1 = \boldsymbol{y}_2$ (convexity in the left parameter) xor $\boldsymbol{x}_1 = \boldsymbol{x}_2$ (in the right parameter).

**Lemma 8.** *For any $t \in \mathbb{R}$, the following holds true:*

**(D)** *$f$ satisfies the data processing inequality iff $f^{(t)}$ satisfies the data processing inequality;*

**(P)** *$\boldsymbol{\mu}_* \in \arg\min_{\boldsymbol{\mu}} \sum_i f(\boldsymbol{x}_i, \boldsymbol{\mu})$ iff*

$$\boldsymbol{\mu}_* \quad \in \quad \arg\min_{\boldsymbol{\mu}} (\oplus_t)_i \, f^{(t)}(\boldsymbol{x}_i, \boldsymbol{\mu}); \qquad (28)$$

**(J)** *$f$ satisfies (27) iff the following $(t, t', t'')$-joint convexity property holds:*

$$f^{(t)}(\lambda \cdot \boldsymbol{x}_1 + (1-\lambda) \cdot \boldsymbol{x}_2, \lambda \cdot \boldsymbol{y}_1 + (1-\lambda) \cdot \boldsymbol{y}_2) \leqslant \lambda f^{(t')}(\boldsymbol{x}_1, \boldsymbol{y}_1) + (1-\lambda) f^{(t'')}(\boldsymbol{x}_2, \boldsymbol{y}_2), \quad (29)$$

*with $t' \doteq \min\{t, 1 - \lambda + \lambda t\}$ and $t'' \doteq \min\{t, \lambda + (1-\lambda)t\}$.*

*Proof.* **(D)** and **(P)** are immediate consequences of Lemma 3 (point [1.], main file) and properties of $\log_t$. We prove **(J)**. $\mathfrak{g}_t$ being strictly increasing for any $t$, we get for $t \leqslant 1$

$$
\begin{aligned}
f^{(t)}(\lambda \cdot \boldsymbol{x}_1 + (1-\lambda) \cdot \boldsymbol{x}_2, \lambda \cdot \boldsymbol{y}_1 + (1-\lambda) \cdot \boldsymbol{y}_2) \quad &\leqslant \quad \mathfrak{g}_t \left( \lambda f(\boldsymbol{x}_1, \boldsymbol{y}_1) + (1-\lambda) f(\boldsymbol{x}_2, \boldsymbol{y}_2) \right) \\
&\leqslant \quad \lambda \cdot \mathfrak{g}_t \circ f(\boldsymbol{x}_1, \boldsymbol{y}_1) + (1-\lambda) \cdot \mathfrak{g}_t \circ f(\boldsymbol{x}_2, \boldsymbol{y}_2) \\
&= \quad \lambda f^{(t)}(\boldsymbol{x}_1, \boldsymbol{y}_1) + (1-\lambda) f^{(t)}(\boldsymbol{x}_2, \boldsymbol{y}_2), (30)
\end{aligned}
$$

because $\mathfrak{g}_t$ is convex. If $t > 1$, we restart from the first inequality and remark that

$$
\begin{aligned}
\mathfrak{g}_t \left( \lambda f(\boldsymbol{x}_1, \boldsymbol{y}_1) + (1-\lambda) f(\boldsymbol{x}_2, \boldsymbol{y}_2) \right) \quad &= \quad \mathfrak{g}_t \left( \lambda f(\boldsymbol{x}_1, \boldsymbol{y}_1) \right) \oplus_t \mathfrak{g}_t \left( (1-\lambda) f(\boldsymbol{x}_2, \boldsymbol{y}_2) \right) \\
&\leqslant \quad \mathfrak{g}_t \left( \lambda f(\boldsymbol{x}_1, \boldsymbol{y}_1) \right) + \mathfrak{g}_t \left( (1-\lambda) f(\boldsymbol{x}_2, \boldsymbol{y}_2) \right) \\
&= \quad \lambda \cdot \mathfrak{g}_{1-\lambda+\lambda t)} \circ f(\boldsymbol{x}_1, \boldsymbol{y}_1) + (1-\lambda) \cdot \mathfrak{g}_{\lambda+(1-\lambda)t} \circ f(\boldsymbol{x}_2, \boldsymbol{y}_2) \\
&= \quad \lambda f^{(1-\lambda+\lambda t)}(\boldsymbol{x}_1, \boldsymbol{y}_1) + (1-\lambda) f^{(\lambda+(1-\lambda)t)}(\boldsymbol{x}_2, \boldsymbol{y}_2). \qquad (31)
\end{aligned}
$$

The inequality is due to the fact that $a \oplus_t b = a + b + (1-t)ab \leqslant a + b$ if $ab \geqslant 0$ and $t \geqslant 1$. The last equality holds because, for $t' \doteq 1 - (1-t)b$, we have

$$\log_t \left( a^b \right) = \frac{a^{(1-t)b} - 1}{1 - t} = \frac{1 - t'}{1 - t} \cdot \frac{a^{1-t'} - 1}{1 - t'} = b \cdot \log_{t'}(a).$$

Putting altogether (30) and (31), we get that for any $t \in \mathbb{R}$,

$$
\begin{aligned}
&f^{(t)}(\lambda \cdot \boldsymbol{x}_1 + (1-\lambda) \cdot \boldsymbol{x}_2, \lambda \cdot \boldsymbol{y}_1 + (1-\lambda) \cdot \boldsymbol{y}_2) \\
&\leqslant \quad \lambda f^{(\min\{t, 1-\lambda+\lambda t\})}(\boldsymbol{x}_1, \boldsymbol{y}_1) + (1-\lambda) f^{(\min\{t, \lambda+(1-\lambda)t\})}(\boldsymbol{x}_2, \boldsymbol{y}_2), \qquad (32)
\end{aligned}
$$

as claimed for **(J)**. This ends the proof of Lemma 8. $\qquad\square$

We note that (29) also translates into a property for **(C)**; also, if $t \leqslant 1$, then $t = t' = t''$ in (29).

## C.V The t-self of the Lorentz model of hyperbolic geometry

As an additional example of use, we consider the Lorentz model for a simple illustration in which one creates approximate metricity in the t-self. This $d$-dimensional manifold is embedded in $\mathbb{R}^{d+1}$ via the hyperboloid with constant $-c < 0$ curvature and defined by $\mathbb{H}_c \doteq \{\boldsymbol{x} \in \mathbb{R}^{d+1} : x_0 > 0 \wedge \boldsymbol{x} \circ \boldsymbol{x} = -1/c\}$, with $\boldsymbol{x} \circ \boldsymbol{y} \doteq -x_0 y_0 + \sum_{i=1}^d x_i y_i$ the Lorentzian inner product [27, Chapter 3]. In ML, two distortions are considered on $\mathbb{H}_c$ [17], one of which is the Lorentzian "distance", $d_L(\boldsymbol{x}, \boldsymbol{y}) \doteq -(2/c) - 2 \cdot \boldsymbol{x} \circ \boldsymbol{y}$. It is notoriously not a distance because it does not satisfy **(T)**, yet we show that we can pick $t$ and the curvature in such a way that the t-self is arbitrarily close to a metric space.

**Lemma 9.** $\forall \delta > 0$, pick $t$ and curvature $c$ as $t = 1 + (1/\delta), c = 2/\delta$. Then the t-self of $\mathbb{H}_c$ is approximately metric: $d_L^{(t)}$ satisfies **(R)**, **(S)**, **(I)**, and the $\delta$-triangle inequality,

$$d_L^{(t)}(\boldsymbol{x}, \boldsymbol{z}) \leqslant d_L^{(t)}(\boldsymbol{x}, \boldsymbol{y}) + d_L^{(t)}(\boldsymbol{y}, \boldsymbol{z}) + \delta, \forall \boldsymbol{x}, \boldsymbol{y}, \boldsymbol{z} \in \mathbb{H}_c. \tag{33}$$

*Proof.* $d_L^{(t)}$ still obviously satisfies reflexivity, the identity of indiscernibles, and non-negativity, so we check the additional property it now satisfies, the weaker version of the triangle inequality. Given any $\boldsymbol{x}, \boldsymbol{y}, \boldsymbol{z}$ in $\mathbb{H}_c$, condition $d_L^{(t)}(\boldsymbol{x}, \boldsymbol{z}) \leqslant d_L^{(t)}(\boldsymbol{x}, \boldsymbol{y}) + d_L^{(t)}(\boldsymbol{y}, \boldsymbol{z}) + \delta$ for $t > 1$ is

$$\frac{1 - \exp\left(-(t-1)\left(-\frac{2}{c} - 2 \cdot \boldsymbol{x} \circ \boldsymbol{z}\right)\right)}{t - 1} \quad \leqslant \quad \frac{1 - \exp\left(-(t-1)\left(-\frac{2}{c} - 2 \cdot \boldsymbol{x} \circ \boldsymbol{y}\right)\right)}{t - 1}$$

$$+ \frac{1 - \exp\left(-(t-1)\left(-\frac{2}{c} - 2 \cdot \boldsymbol{y} \circ \boldsymbol{z}\right)\right)}{t - 1} + \delta,$$

which simplifies to

$$\exp\left(2(t-1) \cdot \boldsymbol{x} \circ \boldsymbol{y}\right) + \exp\left(2(t-1) \cdot \boldsymbol{y} \circ \boldsymbol{z}\right) \quad \leqslant \quad \exp\left(-\frac{2(t-1)}{c}\right) + (t-1)\delta \exp\left(-\frac{2(t-1)}{c}\right)$$

$$+ \exp\left(2(t-1) \cdot \boldsymbol{x} \circ \boldsymbol{z}\right).$$

We have $\boldsymbol{x} \circ \boldsymbol{y} \leqslant -1/c$ by definition, so a sufficient condition to get the inequality is to have

$$\exp\left(2(t-1) \cdot \boldsymbol{y} \circ \boldsymbol{z}\right) \quad \leqslant \quad (t-1)\delta \exp\left(-\frac{2(t-1)}{c}\right). \tag{34}$$

Function $h(z) \doteq z\delta \exp(-2z/c)$ is maximum over $\mathbb{R}_+$ for $z_* = c/2$, for which it equals $h(z_*) = c\delta/(2e)$. Fix $t = 1 + (c/2)$. We then have $\exp\left(2(t-1) \cdot \boldsymbol{y} \circ \boldsymbol{z}\right) \leqslant 1/e$ so to get (34) for this choice of $t$, it is sufficient to pick curvature $c = 2/\delta$, yielding relationship $t = 1 + (1/\delta)$. $\qquad\square$

## C.VI   Proof of Lemma 1

We start by proving condition

$$d_*^{(t)} \quad \geqslant \quad g(k) \tag{35}$$

in the Lemma. Using the proof of [19, Proposition 3.1], we know that any point $\boldsymbol{x}$ $k$-close to the boundary (Definition 5.1) satisfies

$$d^{(t)}(\boldsymbol{x}, \boldsymbol{0}) \quad = \quad \log_t\left(\frac{1 + \|\boldsymbol{x}\|}{1 - \|\boldsymbol{x}\|}\right) = \log_t\left(2 \cdot 10^k - 1\right),$$

so to get (35), we want $\log_t(2 \cdot 10^k - 1) \geqslant g(k)$. Letting $t = 1 - f(k)$ with $f(k) \in \mathbb{R}$, we observe $\log_t(2 \cdot 10^k - 1) = \left((2 \cdot 10^k - 1)^{f(k)} - 1\right)/f(k)$, so we want, after taking logs,

$$\log(2 \cdot 10^k - 1) \quad \geqslant \quad \frac{\log\left(1 + f(k)g(k)\right)}{f(k)} \tag{36}$$

(this also holds if $f(k) < 0$ because $1 + f(k)g(k) < 1$), and there remains to observe $\log(2 \cdot 10^k - 1) = k \log(10) + \log(2 - 1/10^k)$ with $\log(2 - 1/10^k) \geqslant 0, \forall k \geqslant 0$. Hence, to get (36) it is sufficient to request

$$\frac{\log\left(1 + f(k)g(k)\right)}{f(k)} \quad \leqslant \quad \log(10) \cdot k,$$

which is (35). For such $t$, the new hyperbolic constant satisfies

$$\tau_t \quad = \quad \log_t \exp \tau = \frac{1}{f(k)} \cdot \left(\exp\left(f(k)\tau\right) - 1\right)$$

## C.VII   Proof of Theorem 1

Take any observation $\boldsymbol{x} \in \mathcal{X}$. Trivially, $H'(\boldsymbol{x})$ is a real value that belongs to the path followed by $\boldsymbol{x}$ in $H$. Furthermore, if we consider any path from the root to a leaf in $H$, *all* the vertices in its strictly monotonically increasing subsequence of absolute confidences appear in $H'$ (conditions 4, 8 in GETMDT), which guarantees the condition on prediction in **(M)**. Hence **(M)** is satisfied.

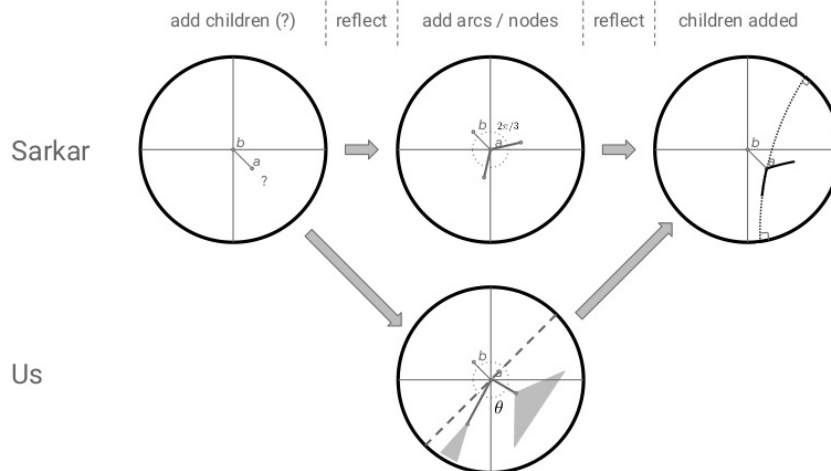

Figure I: Schematic description of our modification (bottom) of Sarkar's algorithm (up). Our modification solely changes the step in which the children of a node (here, $a$) are computed, this node having been reflected back to the origin. Instead of using a fixed angle and length to position arcs (and thus children of the node), the angle depends on the number of leaves that can be reached from a child, and the length depends on the difference between absolute confidence between the node and the corresponding child. We also define an angle which represents the domain (before reflecting back) in which the embedding is going to take place, shown with the thick dashed line (here, this angle is $\pi$).

## C.VIII   Modifying Sarkar's embedding in Poincaré disk

Sarkar's algorithm [30] gives a clean low-distortion embedding when the tree is binary or the arc length is constant [29]. Things are different in our case: MDT nodes have arbitrary out-degrees, and lengths depend on the absolute confidence at the corresponding MDT nodes. Plus, a direct implementation of Sarkar's algorithm would violate the constraint for strict path-monotonicity in an MDT that nodes in a path from the root need to progressively come closer to the border – equivalently, it would create substantial embedding errors for confidences and violate **(B)**. Without focusing on an optimal solution (that we leave for future work), one can remark that all these problems can be heuristically addressed by changing one step of Sarkar's algorithm, replacing the use of the total $2\pi$ fan out angle for mapping children (Step 5: in Algorithm 1 of [29]) by a variable angle with a special orientation in the disk.

We refer to the concise and neat description of Sarkar's embedding in [29] for the full algorithm. Our modification relies on changing one step of the algorithm, as described in Figure I. The key step that we change is step 5: in the description of [29, Algorithm 1]. This step embeds the children of a given node (and then the algorithm proceeds recursively until all nodes are processed). Sarkar's algorithm corresponds to the simple case where all arcs to/from a node define a fixed angle, which does not change during reflection because Poincaré model is conformal. Hence, if the tree is binary, this angle is $2\pi/3$, which provides a very clean display of the tree. In our case however, some children may have just one arc to a leaf while others may support big subtrees. Also, arc lengths can vary substantially. We thus design the region of the disk into which the subtrees are going to be embedded by choosing an angle proportional to the number of leaves reachable from the node, and of course lengths have to match the difference between absolute confidence between a node and its children. There is no optimization step to learn a clean embedding, so we rely on a set of hyperparameters to effectively compute this new step of the algorithm.

## C.IX    Boosting with the logistic loss *à-la* AdaBoost

---

**Algorithm 2** LOGISTICBOOST($\mathcal{S}$)

---

**Input:** Labeled sample $\mathcal{S} \doteq \{(\boldsymbol{x}_i, y_i), i \in [m]\}$, $T \in \mathbb{N}_{>0}$;

1 : $w_{1i} \leftarrow 1/2, \forall i \in [m]$; // initialize all weights (equivalent to maximally *un*confident prediction)

2 :     **for** $j = 1, 2, ..., T$

3 :         $H_j \leftarrow$ WEAK-LEARN($\mathcal{S}, \boldsymbol{w}_j$);                 // request a DT as a "weak hypothesis"

4 :         $\alpha_j \in \mathbb{R}$;                                     // picks leveraging coefficient for $H_j$

5 :         **for** $i = 1, 2, ..., m$                              // weight update, *not* normalized

$$w_{(j+1)i} \quad \leftarrow \quad \frac{w_{ji}}{w_{ji} + (1 - w_{ji}) \cdot \exp(\alpha_j y_i H_j(\boldsymbol{x}_i))}; \tag{37}$$

**Output:** Classifier $\boldsymbol{H}_T \doteq \sum_j \alpha_j H_j(.)$;

---

The link between distances in the Poincaré model of hyperbolic geometry and the canonical link of the log-loss (3) can be extended to leveraging coefficients in boosted combinations [31]. To get there, we need a tailored boosting algorithm which parallels AdaBoost's design tricks (hence different from LogitBoost [13]). We want a boosting algorithm for the liner combination of DTs which displays classification that can be easily and directly embedded in the Poincaré disk. Algorithm LOGISTICBOOST is provided above. For the weight update, we refer to [23]. We already know how to embed DTs via their Monotonic DTs. What a boosting algorithm of this kind does is craft

$$\boldsymbol{H}_T \quad \doteq \quad \sum_{j=1}^T u_j(.), \quad u_j(\boldsymbol{x}) \doteq \alpha_j \cdot H_j(\boldsymbol{x}); \tag{38}$$

Remark that we have merged the leveraging coefficient and the DTs' outputs $H_.$, on purpose. To compute the leveraging coefficients $\alpha_.$ in Step 4, we use AdaBoost's secant approximation trick[†], applied not to the exponential loss but to the logistic loss: for any $z \in [-R, R]$ and $\alpha \in \mathbb{R}$,

$$\log(1 + \exp(-\alpha z)) \quad \leqslant \quad \frac{1+u}{2} \cdot \log(1 + \exp(-\alpha R)) + \frac{1-u}{2} \cdot \log(1 + \exp(\alpha R)). \tag{39}$$

Hence, to compute the leveraging coefficient $\alpha_j$ that approximately minimizes the current loss, $\sum_i w_{ji} \log(1 + \exp(-\alpha y_i H_j(\boldsymbol{x}_i)))$, we minimize instead the upperbound using (39). Letting

$$(\psi_{\text{LOG}})_j^* \quad \doteq \quad \max_{\lambda \in \Lambda(H_j)} \left| \log \left( \frac{p_{j\lambda}^+}{1 - p_{j\lambda}^+} \right) \right| \tag{40}$$

(note the slight abuse of notation for readability, since we write $(\psi_{\text{LOG}})_j^*$ instead of $|(\psi_{\text{LOG}})_j^*|$; we remind that $\Lambda(.)$ denotes the set of leaves of a tree; the index in the local proportion of positive example $p_{j\lambda}^+$ reminds that weights used need to be boosting's weights) which we note can be directly approximated on the Poincaré disk by looking at the leaf nearest to the border of the disk. Because the maximal absolute confidence in the DT is also the maximal absolute confidence in its MDT, we obtain the sought minimum,

$$\alpha_j \quad = \quad \frac{1}{(\psi_{\text{LOG}})_j^*} \cdot \log \left( \frac{1 + r_j}{1 - r_j} \right),$$

where $r_j \in [-1, 1]$ is the normalized edge

$$r_j \quad \doteq \quad \frac{1}{\sum_i w_{ji}} \cdot \sum_{i \in [m]} w_{ji} \cdot \frac{y_i H_j(\boldsymbol{x}_i)}{\max_k |H_j(\boldsymbol{x}_k)|}$$

$$= \quad \mathbb{E}_{i \sim \tilde{w}_j} \left[ \frac{1}{(\psi_{\text{LOG}})_j^*} \cdot \log \left( \frac{p_{j\lambda(\boldsymbol{x}_i)}^+}{1 - p_{j\lambda(\boldsymbol{x}_i)}^+} \right) \right], \tag{41}$$

---

[†]Explained in [31, Section 3.1].

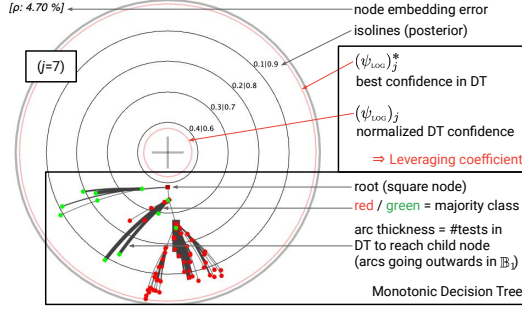

Figure II: One MDT learned on UCI domain `online_shoppers_intention` reproducing the left of Figure 2 in the main file, with additional details for leveraging coefficients.

where $\tilde{\boldsymbol{w}}_j$ indicates normalized weights. It is not hard to show that because we use the local posterior $p_{j\lambda}^+$ at each leaf, $\alpha_j \geqslant 0$ and also $r_j \geqslant 0$. Hence, everything is like if we had an imaginary node $\nu_j$ with $p_{j\nu_j}^+ \doteq (1 + r_j)/2 \ (\geqslant 1/2)$ and positive confidence

$$(\psi_{\text{LOG}})_j \quad \doteq \quad \psi_{\text{LOG}}(p_{j\nu_j}^+) = \log\left(\frac{p_{j\nu_j}^+}{1 - p_{j\nu_j}^+}\right)$$

that we can display in Poincaré disk (we choose to do it as a circle; see Figure II, main file). We deduce from (38) that

$$u_j(\boldsymbol{x}) \quad = \quad \frac{(\psi_{\text{LOG}})_j}{(\psi_{\text{LOG}})_j^*} \cdot \log\left(\frac{p_{j\lambda(\boldsymbol{x})}^+}{1 - p_{j\lambda(\boldsymbol{x})}^+}\right),$$

and note that *all* three key parameters can easily be displayed or computed directly from the Poincaré disk:

- If we use MDTs directly for prediction, $\log\left(\frac{p_{j\lambda(\boldsymbol{x})}^+}{1 - p_{j\lambda(\boldsymbol{x})}^+}\right) = H_j(\boldsymbol{x})$ is read directly from the MDT's plot; the absolute value can be removed and the true sign guessed from the node's color (or shape); if se use DTs for prediction, the MDT's plot gives an approximation of the DT's output which is $100\%$ accurate for all leaves of the MDT (each corresponds to a leaf in the DT);

- If $H_j$ is leveraged (in an ensemble), $(\psi_{\text{LOG}})_j^*$ is read directly from the plot (40), as well as $(\psi_{\text{LOG}})_j$. Their ratio gives the leveraging coefficient. We note that $(\psi_{\text{LOG}})_j^*$ is the same for both the DT and its corresponding MDT. Only the value of $(\psi_{\text{LOG}})_j$ can differ for the DT and the MDT.

We note that regardless of whether we use the DT or its MDT to classify, the MDT's representation is also fully accurate for the DT for a subset of its leaves.

# D    Supplementary material on experiments

## D.I    Domains

The domains we consider are all public domains, from the UCI repository of ML datasets [12], OpenML, or Kaggle, see Table I.

## D.II    Visualizing a DT *via* its MDT and embedding errors

All test errors are estimated from a 10-fold stratified cross-validation. One can always choose to directly learn Monotonic DTs instead of DTs, given their natural fit for embedding in the Poincaré disk. In this case, the hyperbolic representation of the MDT can be immediately used for assessment. Suppose we stick to learning DTs (because *e.g.* they have been standard in ML for decades) *and* wish

| Domain | $m$ | $d$ | License |
|---|---|---|---|
| breastwisc | 699 | 9 | CC BY 4.0 |
| ionosphere | 351 | 33 | CC BY 4.0 |
| tictactoe | 958 | 9 | PDDL |
| winered | 1 599 | 11 | CC BY 4.0 |
| german | 1 000 | 20 | CC BY 4.0 |
| analcatdata_supreme | 4 053 | 8 | CC BY 4.0 |
| abalone | 4 177 | 8 | CC BY 4.0 |
| qsar | 1 055 | 41 | CC BY 4.0 |
| hillnoise | 1 212 | 100 | CC BY 4.0 |
| firmteacher | 10 800 | 16 | CC BY 4.0 |
| online_shoppers_intention | 12 330 | 17 | CC BY 4.0 |
| give_me_some_credit | 120 269 | 11 | None |
| Buzz_in_social_media (Tom's hardware) | 28 179 | 96 | CC BY 4.0 |
| Buzz_in_social_media (twitter) | 583 250 | 78 | CC BY 4.0 |

Table I: UCI, OpenML (Analcatdata_supreme) and Kaggle (Give_me_some_credit) domains considered in our experiments ($m$ = total number of examples, $d$ = number of features), ordered in increasing $m \times n$. Datset licenses listed in last column.

| domain | DT | MDT | p-val |
|---|---|---|---|
| breastwisc | $4.15 \pm 2.18$ | $5.00 \pm 2.38$ | **0.2408** |
| ionosphere | $5.71 \pm 2.70$ | $9.11 \pm 5.34$ | 0.0237 |
| tictactoe | $2.61 \pm 1.85$ | $2.09 \pm 1.10$ | **0.1390** |
| winered | $18.39 \pm 2.02$ | $18.32 \pm 2.27$ | **0.9227** |
| german | $24.00 \pm 4.37$ | $23.90 \pm 4.25$ | **0.8793** |
| analcatdata_supreme | $23.40 \pm 1.85$ | $22.56 \pm 1.75$ | 0.0134 |
| abalone | $22.15 \pm 2.20$ | $21.14 \pm 2.44$ | **0.1267** |
| qsar | $12.98 \pm 2.63$ | $12.89 \pm 4.05$ | **0.8772** |
| hillnoise | $37.04 \pm 2.73$ | $45.88 \pm 4.72$ | 0.0008 |
| firmteacher | $6.87 \pm 1.23$ | $7.04 \pm 1.17$ | **0.4292** |
| online_shoppers_intention | $9.90 \pm 0.78$ | $10.67 \pm 0.54$ | 0.0057 |
| hardware | $1.33 \pm 0.27$ | $1.44 \pm 0.19$ | **0.0534** |

Table II: Results of the experiments checking whether "reducing" the interpretation of a DT to that of its MDT (using its hyperbolic embedding) can give accurate information about the tree as well. Numerical columns, from left to right, give the average$\pm$std dev error for DTs and MDTs and provide the p-value of a Student paired t-test with H0 being the identity of the average errors. Entries in **bold faces** correspond to *keeping H0* for a 0.05 first-order risk. See text for details.

to use the visualization of its corresponding MDT to make inferences on the original DT itself. Can we make reliable conclusions? We explore this question by observing that the prediction from DT to MDT can only change if the leaf $\lambda$ that an example reaches in the DT satisfies two conditions: (a) $\lambda$ does not appear as a leaf in the MDT, and (b) it is tagged to an internal node of the MDT with a confidence of the opposite sign (this is **(D1)**, Step 3: in GETMDT). Without tackling the formal aspect of this question, we have designed a simple experiment for a simple assessment of whether / when this is reasonable. We have trained a set of $T$=200 boosted DTs, each with maximal size 200 (total number of nodes). After having computed the corresponding MDTs, we have compared the test errors of the boosted set of trees and that of the ensemble where each DT is replaced by its MDT, *but* the leveraging coefficients do not change. Intuitively, if test errors are on par, the variation in classification of DTs vs MDTs (including confidences) is negligible, and we can "reduce" the interpretation of the DT to that of its MDT in the Poincaré disk. The results are summarized in Table II.

From Table II, we can safely say that our hypothesis reasonably holds in many cases, with two important domains for which it does not: `ionosphere` and `hillnoise`. For the former domain, we attribute it to the small size of the domain, which prevents training big enough trees; for the latter domain, we attribute it to the fact that the domain contains substantial noise, which makes it

| domain | MDT # in ensemble + visualization error $\rho$ | | | | | | | | | |
|---|---|---|---|---|---|---|---|---|---|---|
| | #1 | #2 | #3 | #4 | #5 | #6 | #7 | #8 | #9 | #10 |
| online_shoppers_intention | 5.42 | 3.74 | 5.46 | 6.42 | 7.00 | 7.96 | 4.70 | 5.68 | 7.93 | 6.22 |
| analcatdata_supreme | 7.11 | 8.32 | 8.23 | 8.67 | 7.60 | 6.15 | 7.85 | 6.55 | 6.82 | 6.63 |
| abalone | 9.37 | 6.53 | 8.20 | 7.61 | 8.18 | 8.20 | 5.91 | 5.88 | 6.51 | 5.43 |
| winered | 12.13 | 11.87 | 8.56 | 6.31 | 4.44 | 3.13 | 2.09 | 1.43 | 3.84 | 3.03 |
| qsar | 16.78 | 12.97 | 12.86 | 5.77 | 4.24 | 2.55 | 1.68 | 2.92 | 2.20 | 4.13 |
| german | 14.74 | 10.06 | 7.73 | 5.05 | 2.70 | 2.36 | 1.26 | 1.19 | 2.07 | 1.57 |
| ionosphere | 15.21 | 9.84 | 8.20 | 6.88 | 3.45 | 5.49 | 8.34 | 4.17 | 5.86 | 8.82 |
| hardware | 7.78 | 8.03 | 3.44 | 3.49 | 2.48 | 1.10 | 0.74 | 0.81 | 0.55 | 0.45 |
| kaggle | 4.92 | 5.84 | 6.24 | 7.07 | 7.87 | 8.06 | 8.06 | 8,70 | 8.89 | 7.89 |

Table III: Embedding error $\rho$ (5) for the visualization of the first ten MDTs in an ensemble learned.

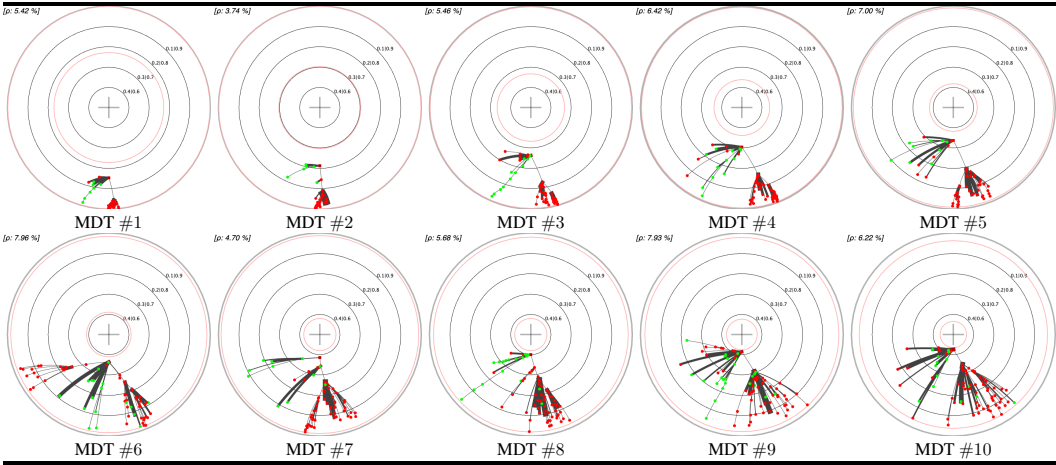

Table IV: First 10 Monotonic Decision Trees (MDTs) embedded in Poincaré disk, corresponding to several Decision Trees (DTs) with $\leqslant 200$ nodes each, learned by boosting the log / logistic-loss on UCI `online_shoppers_intention`. Isolines correspond to the node's prediction confidences, and geometric embedding errors ($\rho\%$) are indicated. See text for details.

difficult to substantially improve posteriors by splitting and thus make many DT nodes, including leaves, "disappear" in the MDT conversion (Steps 2: and 14: in GETMDT). Table III summarizes the visualization errors of the MDT embeddings obtained in Tables IV - XII. A key feature of the embeddings is that on some domains (*e.g.* german) the error of the embedding dramatically decreases with the index of the MDT in the ensemble. The reason why this happens is simple: the weight update of boosting tends to equalize weights between classes (to make the problem "harder"); thus, new trees tend to have their roots embedded closer to the origin, which makes the embedding easier for the rest of the trees to keep good visualization at low error.

### D.III    All Poincaré disk embeddings

They are presented in Table IV through to XII, showing the first models learned in a fold of the cross-validation experiments. producing Table II

### D.IV    Interest of the t-self for visualization

We now test the experimental impact of switching to the t-self in Poincaré disk model (See Table XIII for a clear depiction of how changing $t$ can yields better visualization close to the border of the disk). Recall that switching to the t-self moves way model parts close to $\partial\mathbb{B}$ but keeps a low non-linear distortion around the center, which is thus roughly only affected by a scaling factor. Figure I presents a few results. For domain `online_shoppers_intention`, we note that the part of the tree that is within the isoline defined by posterior $p^+_{\cdot} \in \{0.1, 0.9\}$ gets indeed just scaled: both plots look quite identical. Very substantial differences appear near the border: the best parts of the model could easily

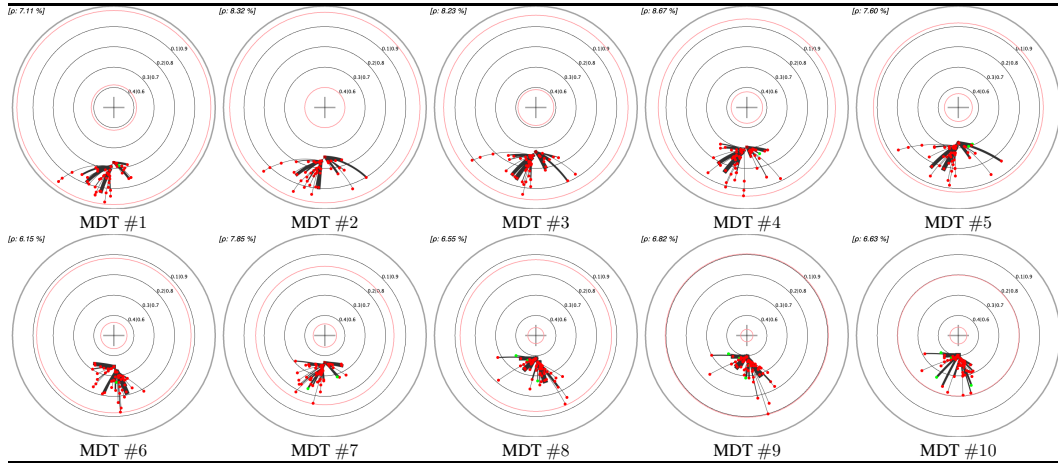

Table V: First 10 MDTs for UCI `analcatdata_supreme`. Convention follows Table IV.

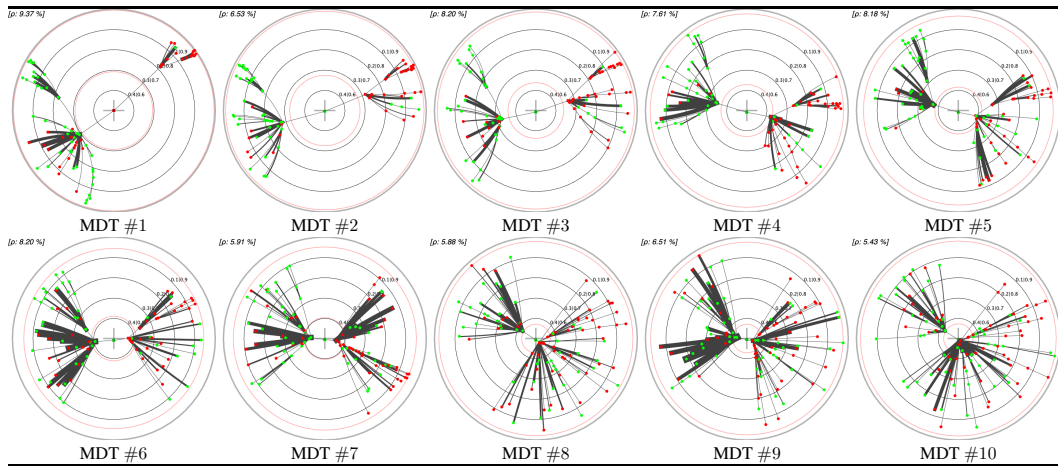

Table VI: First 10 MDTs for UCI `abalone`. Convention follows Table IV.

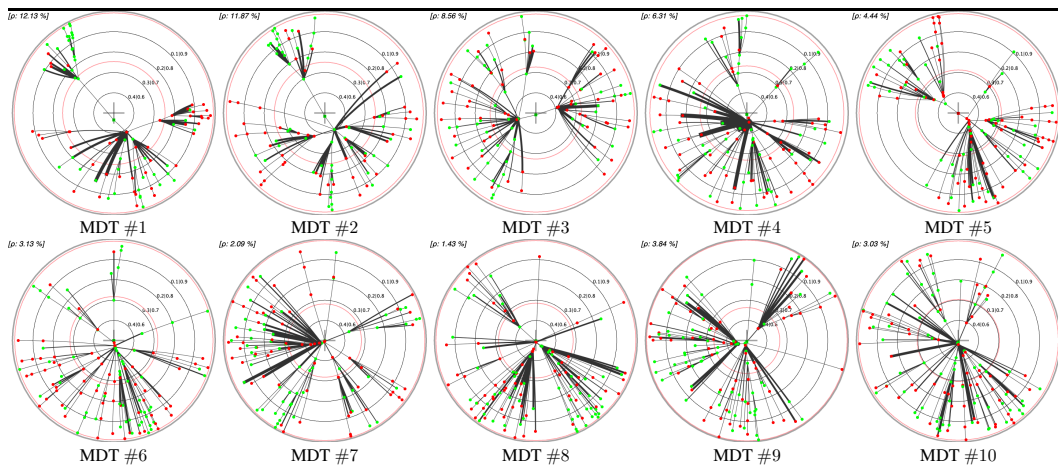

Table VII: First 10 MDTs for UCI `winered`. Convention follows Table IV.

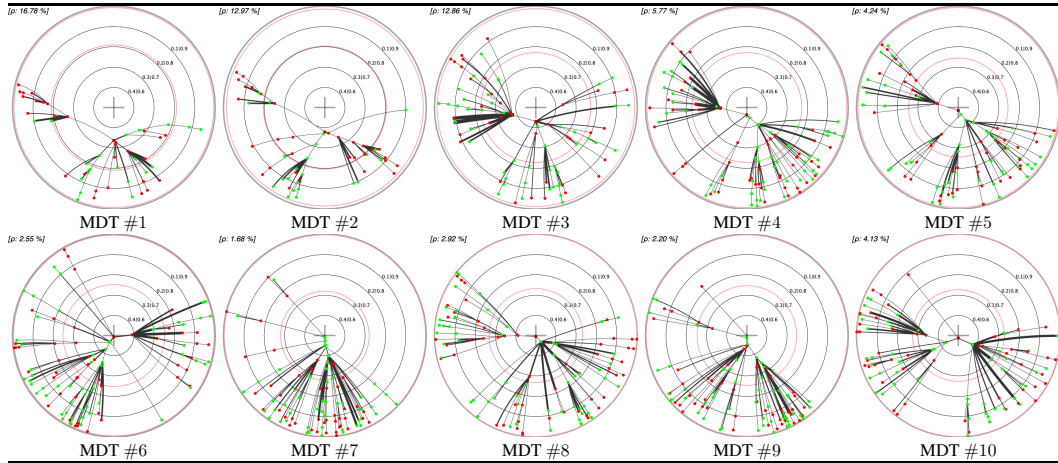

Table VIII: First 10 MDTs for UCI `qsar`. Convention follows Table IV.

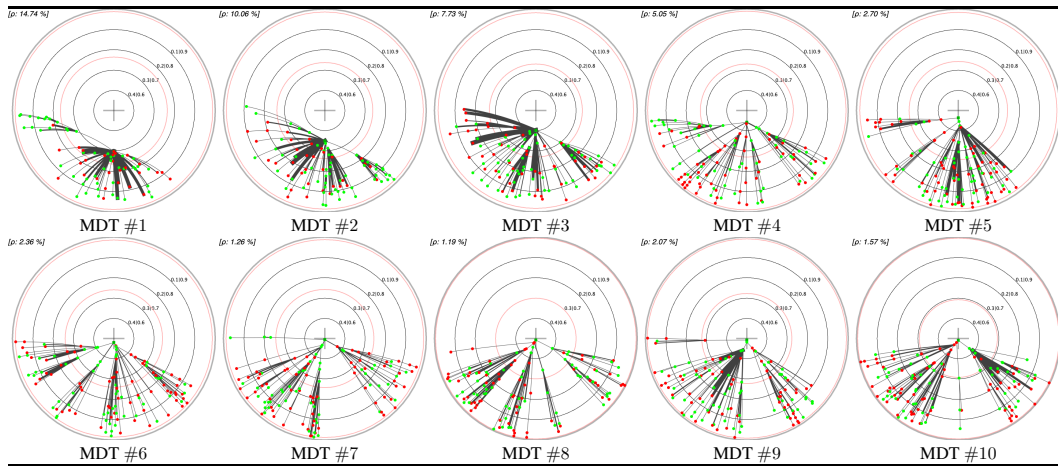

Table IX: First 10 MDTs for UCI `german`. Convention follows Table IV.

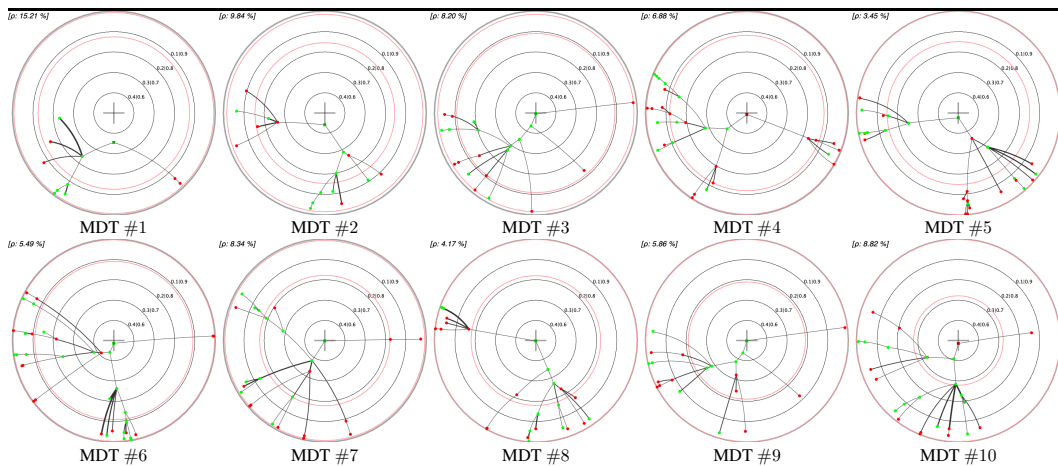

Table X: First 10 MDTs for UCI `ionosphere`. Convention follows Table IV.

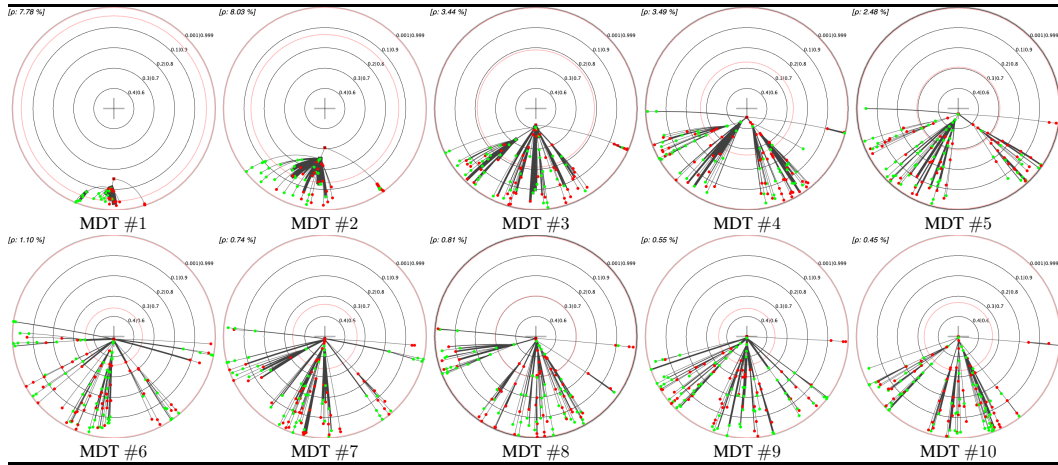

Table XI: First 10 MDTs for UCI `hardware`. Convention follows Table IV.

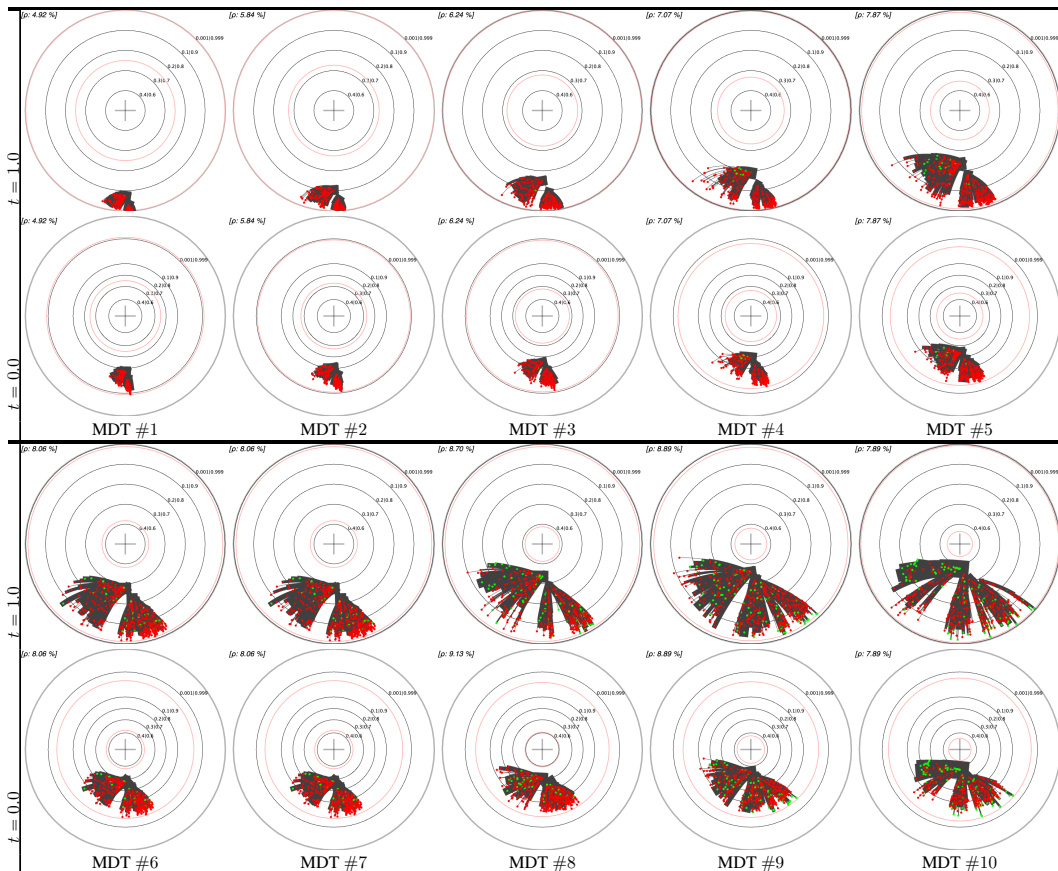

Table XII: Plot of the 10 first MDTs learned on kaggle `give_me_some_credit` (top panel and bottom panel). In each panel, we plot the embedding in $\mathbb{B}$ (top row) and the t-self $\mathbb{B}_1^{(0)}$ ($t = 0$, bottom row). Remark the ability for the t-self to display a clear difference between the best subtrees, subtrees that otherwise appear quite equivalent in terms of confidence from $\mathbb{B}$ alone.

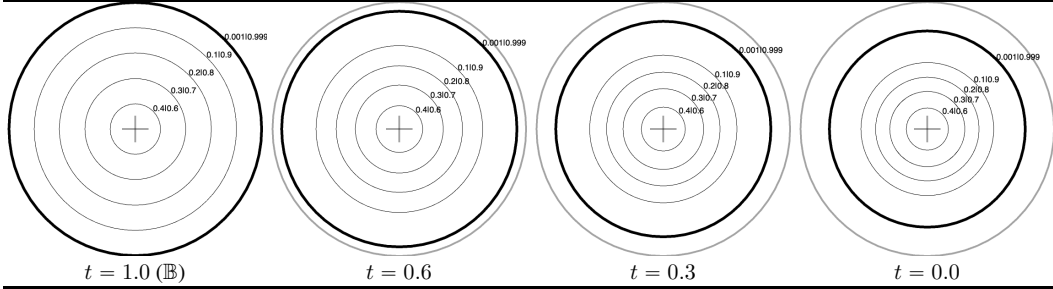

Table XIII: T-self $\mathbb{B}_1^{(t)}$ for Poincaré disk $\mathbb{B}$ (left), for several $t$s. We plot the same set of isolines of $\mathbb{B}$ (and their mapping via $\mathfrak{p}_t$), parameterized by probability $p \in [0, 1]$ which gives the norm $r \doteq |2p - 1|$ in $\mathbb{B}$ (From the innermost to the outermost, $p \in \{0.6, 0.7, 0.8, 0.9, 0.999\}$, or by symmetry for $p' \doteq 1 - p$, also indicated). Remark the equidistance of isolines in $\mathbb{B}$, approximately kept in $\mathbb{B}_1^{(t)}$ for $p$ up to $0.9$ ($p' = 0.1$), while the distortion we need clearly happens near the border: outermost isoline $p \in \{0.001, 0.999\}$ (plotted with **bigger** width) is smoothly and substantially "moved" within the t-self as $t$ decreases, guaranteeing good readability and coding convenience (see text).

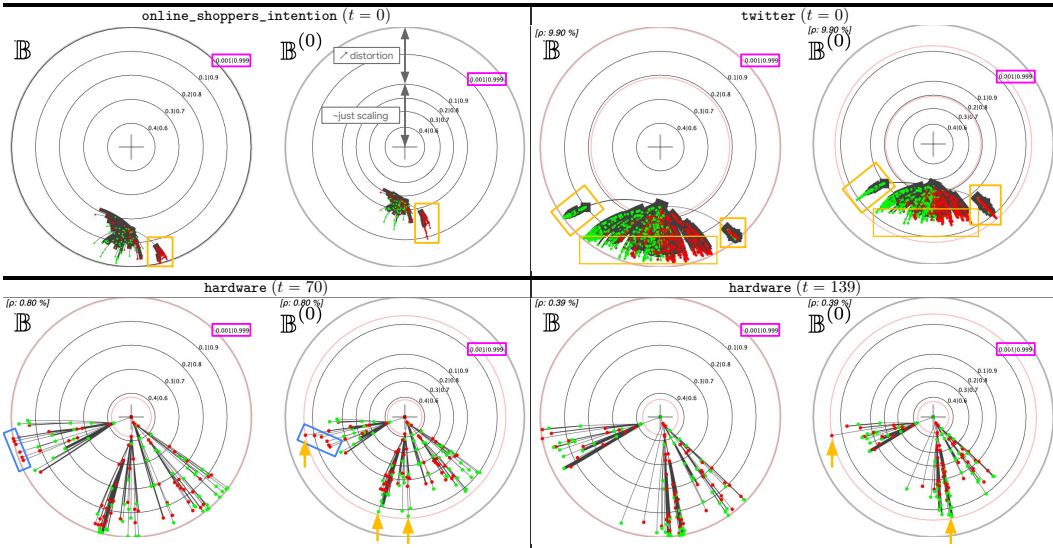

Figure I: *Top pane*: comparison between Poincaré disk embedding and its t-self for $t = 0$ for an MDT learned on UCI `online_shoppers_intention` (left) and `twitter` (right). On the left panel, we have not plotted the boosting coefficients information. The isoline distinguished in magenta is the big **width** one in Table XIII. Note the difference in (non-linear) distortion created in the t-self, in which the central part just enjoys a scaling. *Bottom pane*: stark differences in visualization between $\mathbb{B}$ and the t-self do not just appear initially: they can appear at any iteration. We display two MDTs learned on UCI `hardware` at two different iterations (indicated). It would be very hard to differentiate the best leaves from just the Poincaré disk embedding, while it becomes obvious from the t-self (orange arrows). Note also the set of red nodes in the Poincaré disk for $t = 70$ that mistakenly look aligned, but not in the t-self (blue rectangle). See text for details.

be misjudged as equivalent from $\mathbb{B}$ alone (orange rectangle) but there is little double from $\mathbb{B}_1^{(t)}$ that one of them, which crosses the $p^+ \in \{0.001, 0.009\}$ isoline, is in fact much better than the others. When many subtrees seem to be aggregating near the border as in `buzz_in_social_media`, stark differences can appear on the t-self: the best subtrees are immediately spotted from the t-self (orange rectangles). In between, the t-self makes a much more visible ordering between the best nodes and subtrees, compared to Poincaré disk. `hardware` demonstrate that such very good nodes that are hard to differentiate from the others in $\mathbb{B}$ can appear at any iteration.

